# Gradient Methods for Online DR-Submodular Maximization with Stochastic Long-Term Constraints

**Guanyu Nie**
Iowa State University
Ames, IA 50010
nieg@iastate.edu

**Vaneet Aggarwal**
Purdue University
West Lafayette, IN 47907
vaneet@purdue.edu

**Christopher John Quinn**
Iowa State University
Ames, IA 50010
cjquinn@iastate.edu

## Abstract

In this paper, we consider the problem of online monotone DR-submodular maximization subject to long-term stochastic constraints. Specifically, at each round $t \in [T]$, after committing an action $\mathbf{x}_t$, a random reward $f_t(\mathbf{x}_t)$ and an unbiased gradient estimate of the point $\widetilde{\nabla} f_t(\mathbf{x}_t)$ (semi-bandit feedback) are revealed. Meanwhile, a budget of $g_t(\mathbf{x}_t)$, which is linear and stochastic, is consumed of its total allotted budget $B_T$. We propose a gradient ascent based algorithm that achieves $\frac{1}{2}$-regret of $\mathcal{O}(\sqrt{T})$ with $\mathcal{O}(T^{3/4})$ constraint violation with high probability. Moreover, when first-order full-information feedback is available, we propose an algorithm that achieves $(1 - 1/e)$-regret of $\mathcal{O}(\sqrt{T})$ with $\mathcal{O}(T^{3/4})$ constraint violation. These algorithms significantly improve over the state-of-the-art in terms of query complexity.

## 1 Introduction

*Online optimization* confronts diverse challenges as information gradually unfolds, compelling irreversible decisions at each step amidst uncertainty about future information. Typically, an online optimization problem can be framed as a recurring game between a learner or algorithm and an adversary or environment: during each iteration, the learner chooses an action from a predefined domain set, and the environment subsequently reveals feedback in the form of utility or reward for the selected action. Over time, the learner aims to learn from past experiences and improve decision-making strategies to maximize cumulative rewards.

In instances where the objective function is concave and the feasible set is convex, the problem has been extensively explored in the literature under the name of *online convex optimization* (OCO) [13]. OCO has found considerable success in various machine learning applications, leveraging the well-established theories of convex optimization. Within OCO, it is established that any algorithm incurs a regret of $\Omega(\sqrt{T})$ in the worst case [13]. Notably, there exist algorithms that match this lower bound, such as Online Gradient Descent (OGD) [43].

Even though optimizing convex/concave functions can be done efficiently, most problems in artificial intelligence are non-convex. Examples include training deep neural networks, Bayesian inference, and clustering, among many others. One important example of such functions is called *submodular set function*. Submodular set functions exhibit a natural property known as *diminishing returns*, akin to concave functions in continuous domains. They have been applied in various machine learning contexts, including viral marketing [16], sensor placement [11], recommendation systems [23], data summarization [22], and numerous others. However, submodularity extends beyond set functions and can also be defined for continuous functions. DR-submodularity represents a special class of such continuous functions [5].

In the realm of online DR-submodular maximization, two primary types of algorithms have received extensive attention. Gradient ascent algorithms are often favored for their simplicity and lower gradient sample complexity, while Frank-Wolfe algorithms offer the ability to avoid potentially costly projection operations [9]. The type of feedback also plays a pivotal role in algorithm design. Frank-Wolfe algorithms typically assume more information is acquired each round, assuming that upon committing an action, the agent can observe the entire function, and gradients of multiple points can be observed. This feedback type is often referred to as *full information feedback*. On the other hand, gradient ascent-based algorithms usually assume only the function value of the chosen action and the gradient at that point can be observed. This feedback type is often referred to as *semi-bandit feedback*. Recent efforts have aimed to enhance gradient queries for Frank-Wolfe algorithms, albeit at the cost of worse regret [41].

In numerous applications, aside from the goal of maximizing the cumulative reward, there exist constraints on the sequence of decisions made by the learner that must be satisfied on average [1, 3]. In such scenarios, long-term constraints pose challenges because the decision-making process cannot be simply expressed as restricting the set of actions for all time steps may be sub-optimal. Instead, the learner still (always) wants to maximize cumulative reward, but if putatively instantaneously high reward arms are expensive, can only play them a few times. To illustrate, let's consider the online ad allocation problem encountered by an advertiser. At each round $t \in [T]$, the advertiser faces the task of determining the allocation of funds across $n$ different websites for placing ads. While the primary aim is to maximize the overall click-through rates of the ads, the advertiser is also constrained by a predetermined budget allocated over a specified time horizon [4]. In this scenario, the cost of ad placement in each round may either be fixed and known in advance, or it may depend on the number of clicks the ads receive, which remains uncertain in advance. As a result, the advertiser must navigate the trade-off between optimizing the total reward and adhering to budget constraints.

In this paper, we study the problem of online DR-submodular maximization with long-term (linear) budget constraints. While most prior works assume the objective function is chosen adversarially [10, 29, 31], we study the stochastic DR-submodular maximization setting, where the objective functions are i.i.d. sampled from a distribution [7, 12]. Note that this setting is still of interest because as opposed to assuming each arriving function $f_t$ being DR-submodular, we only assume the expectation of $f_t$ to possess DR-submodularity. Moreover, we consider the semi-bandit feedback setting where only the noisy function value $f_t(\mathbf{x}_t)$ and an unbiased gradient estimator of that point $\widetilde{\nabla} f_t(\mathbf{x}_t)$ can be observed. To the best of our knowledge, these particular settings (stochastic utility and semi-bandit feedback) have not been explored in the literature on DR-submodular maximization with long-term constraints.

**Our Contributions:** We summarize our contributions in three parts.

1. In the semi-bandit feedback setting, we propose the first stochastic gradient ascent based algorithm for stochastic online DR-submodular maximization with stochastic long-term constraints. Our proposed algorithm can achieve $\mathcal{O}(\sqrt{T})$ $\frac{1}{2}$-regret and $\mathcal{O}(T^{3/4})$ constraint violation with high probability. Compared to all previous works [10, 29, 31], they consider first-order full-information feedback and require unbiased gradient estimates at $\sqrt{T}$ locations (not just at action $\mathbf{x}_t$) in every round. For those works, their query complexity in each round is $\sqrt{T}$ while ours is just 1.

2. In first order full-information setting, where the unbiased gradient estimates of any point can be observed, we propose the first stochastic gradient ascent based algorithm for stochastic online DR-submodular maximization with stochastic long-term constraints. We utilize the recently developed technique in [42] called the non-oblivious function. Our proposed algorithm can achieve $\mathcal{O}(\sqrt{T})$ $(1 - 1/e)$-regret and $\mathcal{O}(T^{3/4})$ constraint violation with high probability. Again, compared to previous works [10, 29, 31], our query complexity is significantly lower.

**Regarding the approximation ratios:** We note that in offline $1 - 1/e$ is known to be the optimal approximation ratio for optimizing monotone DR submodular functions over a general convex set, where the query can be anywhere in the convex hull of $\mathcal{K} \cup \{\mathbf{0}\}$ ($\mathcal{K}$ is the constraint set). However, when the oracle calls are restricted to $\mathcal{K}$, an approximation ratio of $1/2$ is the best that is known to be achievable [28]. Thus, full information feedback can achieve $1 - 1/e$-regret while the semi-bandit feedback achieves $1/2$-regret.

Table 1: We include related works from online DR-submodular optimization with constant or stochastic long term constraint functions. (Works handling adversarial long-term constraints require a different definition of regret.) All methods require a gradient oracle for feedback, and 'Noise' lists whether the gradient is exact or there is stochastic noise. '# Grad.' is the number gradient evaluations required per-round. 'Con. Viol.' is the bound on the constraint violation. † [31] considered constraint set being convex while all other works consider linear constraint. In '# Grad.' column, $2\sqrt{T}$ means this work needs $\sqrt{T}$ gradients on both $f$ and $g$. ‡ While all actions will be feasible, some gradient queries will be in the convex hull of $\mathcal{K} \cup \{\mathbf{0}\}$.

| REFERENCE | REGION | NOISE | # GRAD. | APPROX. | REGRET | CON. VIOL. |
|---|---|---|---|---|---|---|
| [29] | $\mathbf{0} \in \mathcal{K}$ | $\times$ | $\sqrt{T}$ | $1-1/e$ | $\mathcal{O}(\sqrt{T})$ | $\widetilde{\mathcal{O}}(\sqrt{T})$ |
| [31]† | $\mathbf{0} \in \mathcal{K}$ | $\times$ | $2\sqrt{T}$ | $1-1/e$ | $\mathcal{O}(\sqrt{T})$ | $\mathcal{O}(\sqrt{T})$ |
| [10] | $\mathbf{0} \in \mathcal{K}$ | $\times$ | $\sqrt{T}$ | $1-1/e$ | $\mathcal{O}(\sqrt{T})$ | $\mathcal{O}(\sqrt{T})$ |
| [29] | $\mathbf{0} \in \mathcal{K}$ | $\checkmark$ | $\sqrt{T}$ | $1-1/e$ | $\mathcal{O}(T^{3/4})$ | $\widetilde{\mathcal{O}}(\sqrt{T})$ |
| THEOREM 2 | GENERAL ‡ | $\checkmark$ | $1$ | $1-1/e$ | $\mathcal{O}(\sqrt{T})$ | $\mathcal{O}(T^{3/4})$ |
| THEOREM 1 | GENERAL | $\checkmark$ | $1$(SEMI) | $1/2$ | $\mathcal{O}(\sqrt{T})$ | $\mathcal{O}(T^{3/4})$ |

## 2 Related Works

The primary related works are summarized in Table 1. We briefly discuss notable contributions and for additional related works, see Appendix F

**Online DR-submodular Maximization with Long-Term Constraints** We do not compare results with adversarial constraints due to additional assumptions needed in this setting. See Appendix F for a detailed discussion. In the context of stochastic constraints, Raut et al. [29] conducted the initial study of the problem. They successfully attained $\mathcal{O}(\sqrt{T})$ regret and constraint violation with high probability, as well as $\mathcal{O}(T^{3/4})$ regret and $\mathcal{O}(\sqrt{T})$ constraint violation in expectation. Building upon this work, Sadeghi et al. [31] further improved the results to achieve $\mathcal{O}(\sqrt{T})$ regret and constraint violation, both in expectation and with high probability. Additionally, Feng et al. [10] extended these findings to incorporate weakly DR-submodular utility, achieving analogous results.

**Online Convex Optimization with Long-Term Constraints** Several results for OCO with deterministic long-term constraints can be found in [14, 20, 37, 38, 40]. Existing literature has established that a regret of $\mathcal{O}(\sqrt{T})$ and a cumulative constraint violation of $\mathcal{O}(T^{1/4})$ can be achieved without the Slater condition. Conversely, assuming the Slater condition allows for achieving a regret of $\mathcal{O}(\sqrt{T})$ and a cumulative constraint violation of $\mathcal{O}(1)$. In cases where the considered constraint is assumed to be stochastic, Yu et al. [39] achieved a $\mathcal{O}(\sqrt{T})$ bound on both regret and constraint violations under the Slater condition. Furthermore, Wei et al. [35] achieved the same regret and constraint violation bounds while assuming a strictly weaker assumption than the Slater condition.

## 3 Preliminaries

### 3.1 Notations

Vectors are shown by lowercase bold letters, such as $\mathbf{x} \in \mathbb{R}^d$. We denote by $\|\cdot\|$ the $\ell_2$ (Euclidean) norm. We use $[T]$ to denote the set $\{1, 2, \dots, T\}$. The inner product of two vectors $\mathbf{x}, \mathbf{y} \in \mathbb{R}^d$ is denoted by either $\langle \mathbf{x}, \mathbf{y} \rangle$ or $\mathbf{x}^\top \mathbf{y}$. For $u \in \mathbb{R}$, we define $[u]_+ := \max\{u, 0\}$. For two vectors $\mathbf{x}, \mathbf{y} \in \mathbb{R}^d$, $\mathbf{x} \preceq \mathbf{y}$ implies that $x_i \leq y_i, \forall i \in [d]$. For a convex set $\mathcal{X}$, we denote the projection of $\mathbf{y}$ onto set $\mathcal{X}$ as $\Pi_{\mathcal{X}}(\mathbf{y}) = \arg\min_{\mathbf{x} \in \mathcal{X}} \|\mathbf{x} - \mathbf{y}\|$.

### 3.2 Function Properties

Here we list some function properties that will appear in our assumptions.

**Monotonicity** A function $f$ is monotone if $f(\mathbf{x}) \leq f(\mathbf{y})$ for all $\mathbf{x} \preceq \mathbf{y}$.

**Lipschitz continuous** A function $f$ is Lipschitz continuous with parameter $\beta$ if for any $\mathbf{x}, \mathbf{y} \in \mathcal{X}$, we have $f(\mathbf{x}) - f(\mathbf{y}) \le \beta \|\mathbf{x} - \mathbf{y}\|$.

### 3.3 DR-Submodular Functions

A function $f : 2^\Omega \to \mathbb{R}_+$, defined on the ground set $\Omega$, is said to be submodular if

$$f(A) + f(B) \ge f(A \cup B) + f(A \cap B)$$

for all $A, B \subset \Omega$. The notion of submodularity has been extended to continuous domains [2, 34, 36]. Consider a function $f : \mathcal{X} \to \mathbb{R}_+$ where the domain is of the form $\mathcal{X} = \prod_{i=1}^n \mathcal{X}_i$ and each $\mathcal{X}_i$ is a compact subset of $\mathbb{R}_+$. We say that $f$ is continuous submodular if $f$ is continuous and for all $\mathbf{x}, \mathbf{y} \in \mathcal{X}$, we have

$$f(\mathbf{x}) + f(\mathbf{y}) \ge f(\mathbf{x} \vee \mathbf{y}) + f(\mathbf{x} \wedge \mathbf{y})$$

where $\mathbf{x} \vee \mathbf{y}$ and $\mathbf{x} \wedge \mathbf{y}$ are component-wise maximum and minimum, respectively. For efficient maximization, we also require that these functions satisfy a diminishing returns condition [5]. We say a differentiable function $f$ is continuous DR-submodular if $f$ it satisfies

$$\nabla f(\mathbf{x}) \succeq \nabla f(\mathbf{y})$$

for all $\mathbf{x} \preceq \mathbf{y}$. When the function $f$ is twice differentiable, DR-submodularity is equivalent to

$$\frac{\partial^2 f(\mathbf{x})}{\partial x_i \partial x_j} \le 0, \forall i, j, \forall \mathbf{x} \in \mathcal{X}.$$

The following lemma is an important property of monotone DR-submodular functions.

**Lemma 1.** *Let $f : \mathcal{X} \to \mathbb{R}_+$ be a monotone, differentiable, DR-submodular function. For any two vectors $\mathbf{x}, \mathbf{y} \in \mathcal{X}$, we have*

$$f(\mathbf{y}) - 2f(\mathbf{x}) \le \langle \nabla f(\mathbf{x}), \mathbf{y} - \mathbf{x} \rangle.$$

The proof can be found in the proof of Theorem 4.2 in [12]. Note that this property shares an analogy with a pivotal characteristic that defines concavity: $f(\mathbf{y}) - f(\mathbf{x}) \le \langle \nabla f(\mathbf{x}), \mathbf{y} - \mathbf{x} \rangle$. Lemma 1 also implies that if we use gradient descent directly on $f$, we can only achieve an approximation ratio of $1/2$. Zhang et al. [42] introduced the so-called non-oblivious function to obtain the optimal approximation ratio of $1 - 1/e$:

**Lemma 2.** *Let $f : \mathcal{X} \to \mathbb{R}_+$ be a monotone, differentiable, DR-submodular function, and let $F$ be the non-oblivious function of $f$ defined by its gradient $\nabla F(\mathbf{x}) = \int_0^1 e^{z-1} \nabla f(z \cdot \mathbf{x}) dz$. Then for any vectors $\mathbf{x}, \mathbf{y} \in \mathcal{X}$, we have*

$$\left(1 - e^{-1}\right) f(\mathbf{y}) - f(\mathbf{x}) \le \langle \nabla F(\mathbf{x}), \mathbf{y} - \mathbf{x} \rangle.$$

The proof of Lemma 2 can be found in [42].

## 4 Problem Statement

Consider the following offline optimization problem:

$$\begin{aligned} \max_{\mathbf{x} \in \mathcal{K}} \quad & f(\mathbf{x}) \\ \text{subject to} \quad & g(\mathbf{x}) \le 0, \end{aligned} \tag{1}$$

where $g(\mathbf{x}) = \langle \boldsymbol{p}, \mathbf{x} \rangle - b$ for some non-negative constant $b$. We study an analogous online setup as follows: At each round $t \in [T]$, the algorithm chooses an action $\mathbf{x}_t \in \mathcal{K}$, where $\mathcal{K} \subset \mathbb{R}_+^d$ is a fixed, known set. We consider both the utility and the constraints being stochastic, where we assume at each time step, the utility function $f_t$ is sampled i.i.d. from a distribution $\mathcal{D}_f$ with mean $f$, i.e., $\mathbb{E}_{f_t \sim \mathcal{D}_f}[f_t(\cdot)] = f(\cdot)$, while the cost vector $\boldsymbol{p}_t$ is i.i.d. sampled from another distribution $\mathcal{D}_p$. After an action is selected by the learner, a random reward $f_t(\mathbf{x}_t)$ is obtained while using $\langle \boldsymbol{p}_t, \mathbf{x}_t \rangle$ of its fixed total allotted budget $B_T$, and $\boldsymbol{p}_t$ is observed. In the semi-bandit setting, an unbiased gradient estimator for that action, $\widetilde{\nabla} f_t(\mathbf{x}_t)$, is also revealed. In the first order full-information setting, the unbiased gradient estimator of any point can be observed. In this paper, we consider both settings while all other works in the literature on DR-submodular maximization with long-term constraints, such as [10, 29, 31], consider full-information feedback.

To make sure the long-term constraint is not vacuous, we consider $B_T = bT$ for a constant $b$ such that $\min_{\mathbf{x}\in\mathcal{K}}\langle \boldsymbol{p}, \mathbf{x}\rangle \leq b < \max_{\mathbf{x}\in\mathcal{K}}\langle \boldsymbol{p}, \mathbf{x}\rangle$. In this case, there will always be a solution $\mathbf{x}$ that satisfies constraint (having zero constraint violation) and it is not the case that any sequence of actions (especially the most expensive w.r.t. $\boldsymbol{p}$) is feasible.

We make the following assumptions to proceed our analysis:

**Assumption 1.** The constraint set $\mathcal{K}$ is convex and compact, with diameter $d = \sup_{\mathbf{x},\mathbf{y}\in\mathcal{K}} \|\mathbf{x} - \mathbf{y}\|$ and radius $r = \sup_{\mathbf{x}\in\mathcal{K}} \|\mathbf{x}\|$. Since $\mathcal{X}$ is compact, we denote its diameter as $\bar{d} = \sup_{\mathbf{x},\mathbf{y}\in\mathcal{X}} \|\mathbf{x} - \mathbf{y}\|$ and radius as $\bar{r} = \sup_{\mathbf{x}\in\mathcal{X}} \|\mathbf{x}\|$, respectively.

**Assumption 2.** The expected utility function $f(\cdot)$ is monotone DR-submodular and $\beta_f$-Lipschitz.

**Assumption 3.** The distribution $\mathcal{D}_p$ for the cost vectors has bounded support $\beta_p\mathcal{B} \cap \mathbb{R}_+^d$ with mean $\boldsymbol{p} \succeq \mathbf{0}$, where $\mathcal{B}$ is the unit ball of Euclidean norm.

**Assumption 4.** The gradient oracle is unbiased $\mathbb{E}[\nabla f(\mathbf{x}) - \widetilde{\nabla} f_t(\mathbf{x})|\mathbf{x}] = 0$ and has a bounded variance $\mathbb{E}[\|\nabla f(\mathbf{x}) - \widetilde{\nabla} f_t(\mathbf{x})\|^2|\mathbf{x}] \leq \sigma^2$. In the semi-bandit setting, we assume $G = \max_t \sup_{\mathbf{x}} \|\widehat{\nabla} f_t(\mathbf{x})\|$ is finite. In the first order full-information setting, denoting the unbiased estimator of the non-oblivious function obtained at round $t$ by $\widetilde{\nabla} F_t(\mathbf{x})$, we assume $G_F = \max_t \sup_{\mathbf{x}} \|\widehat{\nabla} F_t(\mathbf{x})\|$ is finite.

Unlike Frank-Wolfe type algorithms in other papers [10, 29, 31], we do not assume bounded smoothness on the gradients: $\nabla f(\mathbf{x}) - \nabla f(\mathbf{y}) \leq \beta\|\mathbf{x} - \mathbf{y}\|$. Moreover, we do not assume $f(\mathbf{0}) = 0$.

Our overall goal is to maximize the total obtained reward while satisfying the budget constraint asymptotically (i.e., $\sum_{t=1}^{T}\langle \boldsymbol{p}, \mathbf{x}_t\rangle - B_T$ being sub-linear in $T$).

Note that our proposed algorithm can handle multiple linear constraints as well, and similar regret and constraint violation bounds can be derived. In the case of there are $m$ constraints $g_i(\cdot)$, $i \in [m]$, we can define $g(\mathbf{x}) := \max_{i\in[m]} g_i(\mathbf{x})$ and it can be shown that $g$ preserves the same properties as those of individual $g_i$'s (sub-differentiability, bounded (sub-)gradients and bounded values; see Proposition 6 in [20] for proofs).

For simplicity of presentation, we denote $\beta = \max\{\beta_f, \beta_p\}$. Since $\mathcal{K}$ is compact, from monotonicity of $f$ we have $F_1 := \max_{\mathbf{x}\in\mathcal{K}} |f(\mathbf{x})|$ is bounded. Since $f$ is $\beta_f$-Lipschitz, we have $F_2 := \max_{\mathbf{x},\mathbf{y}\in\mathcal{K}} |f(\mathbf{x}) - f(\mathbf{y})| \leq \beta_f D$ is bounded. Since $\mathcal{K}$ is compact and $\mathcal{D}_p$ has bounded support, we have $C := \max_{\boldsymbol{p}'\sim\mathcal{D}_p} \max_{\mathbf{x}\in\mathcal{K}} |\langle \boldsymbol{p}', \mathbf{x}\rangle - \frac{B_T}{T}|$ is bounded.

To measure the effectiveness of our proposed algorithm, we use the notions of *regret* and *total constraint violation* to quantify the overall utility and the total resource consumption, respectively.

**Regret** is typically defined as the difference between the total reward accumulated by the algorithm and the best fixed action in hindsight. Note that even in the offline setting, maximizing a monotone DR-submodular function subject to a convex constraint can only be done approximately in polynomial time unless RP = NP [5]. Thus, we instead use the notion of $\alpha$-*regret* of an algorithm.

**Definition 1.** The $\alpha$-regret of an online algorithm with outputs $\{\mathbf{x}_t\}_{t=1}^{T}$ is defined as

$$R_T := \alpha \max_{\mathbf{x}\in\mathcal{K}^*} \sum_{t=1}^{T} f_t(\mathbf{x}) - \sum_{t=1}^{T} f_t(\mathbf{x}_t), \tag{2}$$

where $\mathcal{K}^*$ is the restricted search space of solutions that satisfy long-term constraints for $T$ steps, (i.e., can be played $T$ times), $\mathcal{K}^* = \{\mathbf{x} \in \mathcal{K} : \sum_{t=1}^{T} g(\mathbf{x}) \leq 0\}$, which is also equivalent as satisfying per-round constraint: $\mathcal{K}^* = \{\mathbf{x} \in \mathcal{K} : g(\mathbf{x}) \leq 0\}$.

Since we are mainly interested in stochastic utility functions, i.e., $f_t \sim \mathcal{D}_f$, we aim to minimize the expected $\alpha$-regret:

$$\mathbb{E}[R_T] = \alpha T \max_{\mathbf{x}\in\mathcal{K}^*} f(\mathbf{x}) - \sum_{t=1}^{T} f(\mathbf{x}_t).$$

Denote $\mathbf{x}^* = \arg\max_{\mathbf{x}\in\mathcal{K}^*} f(\mathbf{x})$. Note that since $\boldsymbol{p}_t$ is drawn i.i.d. from the distribution $\mathcal{D}_p$ with mean $\boldsymbol{p}$ $\forall t \in [T]$, the best benchmark action is with respect to the "true" underlying $\boldsymbol{p}$ of the constraint

function as opposed to $p_t$. It is possible that the best-fixed action has a constraint violation with some noisy $p_t$'s.

We next define the total constraint violation.

**Definition 2.** The *total constraint violation* of an online algorithm with outputs $\{\mathbf{x}_t\}_{t=1}^{T}$ is defined as

$$C_T := \sum_{t=1}^{T} g(\mathbf{x}_t) = \sum_{t=1}^{T} \langle \boldsymbol{p}, \mathbf{x}_t \rangle - B_T.$$

Again, in the stochastic constraint setting, the total constraint violation is defined with respect to the mean $\boldsymbol{p}$.

## 5 An Efficient Primal-Dual Algorithm under Semi-bandit Feedback

In this section, we introduce our first proposed algorithm for online DR-submodular maximization subject to stochastic long term constraints under semi-bandit feedback: Online Lagrangian Stochastic Gradient Ascent (OLSGA). The algorithm is presented in Algorithm 1. The overall structure of the algorithm is inspired by the primal-dual update in Online Convex Optimization (OCO) (e.g., [20]).

Associating a dual variable $\lambda \in [0, +\infty)$ with the constraint, the saddle point formulation of (1) can be written as

$$\max_{x \in \mathcal{K}} \min_{\lambda \in [0, +\infty)} f(\mathbf{x}) - \lambda g(\mathbf{x}).$$

In this work, we consider the following regularized Lagrangian function $\mathcal{L}(\mathbf{x}, \lambda)$ given by

$$\mathcal{L}(\mathbf{x}, \lambda) := f(\mathbf{x}) - \lambda g(\mathbf{x}) + \frac{\delta\eta}{2}\lambda^2. \tag{3}$$

It is important to observe that the expression in (3) deviates from the conventional Lagrangian due to the inclusion of the term $\frac{\delta\eta}{2}\lambda^2$, where both $\delta$ and $\eta$ are parameters that will be later chosen to optimize theoretical guarantees. The main purpose of this modification is to control the value of $\lambda$ and prevent it from growing too large. Although we can achieve the same goal by restricting $\lambda$ to a bounded domain, using the quadratic regularizer makes it convenient for our analysis.

One issue is that $\boldsymbol{p}$ (shown in $g(\mathbf{x})$) is unknown to the online algorithm. Therefore, we alternatively use an empirical estimate $\widehat{\boldsymbol{p}}_t = \frac{1}{t}\sum_{s=1}^{t} \boldsymbol{p}_s$ instead of $\boldsymbol{p}$ in the Lagrangian function. Moreover, in order to achieve the high probability bound, we adjust our Lagrangian function in (3) as follows:

$$\mathcal{L}_t(\mathbf{x}, \lambda) = f_t(\mathbf{x}) - \lambda \widetilde{g}_t(\mathbf{x}) + \frac{\delta\eta}{2}\lambda^2, \tag{4}$$

where $\widetilde{g}_t(\mathbf{x}) = \langle \widehat{\boldsymbol{p}}_t, \mathbf{x} \rangle - \frac{B_T}{T} - \gamma_t$ and $\gamma_t = \sqrt{\frac{2C^2 \log(\frac{2T}{\varepsilon})}{t}}$. For the purpose of analysis, we further define $\widehat{g}_t(\mathbf{x}) := \langle \widehat{\boldsymbol{p}}_t, \mathbf{x} \rangle - \frac{B_T}{T}$.

For the purpose of analysis, we do not directly use Equation (4) in our primal update. Let $\widehat{\mathcal{L}}_t$ be defined by its gradient, $\widetilde{\nabla}_x \widehat{\mathcal{L}}_t(\mathbf{x}_t, \lambda_t) = \widetilde{\nabla} f_t(\mathbf{x}_t) - 2\lambda_t \nabla \widetilde{g}_t(\mathbf{x}_t)$. The primal updates are formulated as follows:

$$\mathbf{x}_{t+1} = \Pi_{\mathcal{K}}(\mathbf{x}_t + \eta \widetilde{\nabla}_x \widehat{\mathcal{L}}_t(\mathbf{x}_t, \lambda_t)).$$

Note that compared to Equation (4), the Lagrangian function used for updating has a coefficient of 2 in front of the second term.

Our proposed algorithm is shown in Algorithm 1. The algorithm proceed as follows: it takes a convex constraint set $\mathcal{K}$ and a time horizon $T$ as inputs. Initially, the algorithm selects an initial point $\mathbf{x}_1 \in \mathcal{K}$ and sets $\lambda_1 = 0$. At each time step $t \in [T]$, the algorithm takes an action $\mathbf{x}_t$, acquires a reward $f_t(\mathbf{x}_t)$, and observes the cost vector $\boldsymbol{p}_t$ as well as an unbiased gradient estimate $\widetilde{\nabla} f_t(\mathbf{x}_t)$. Subsequently, unbiased gradient estimates of the updating Lagrangian function with respect to $\mathbf{x}$ and to $\lambda$ are computed using the empirical estimate of $\boldsymbol{p}$. Using these calculated gradients, updates to $\mathbf{x}$ and $\lambda$ are made using (7) and (8), respectively.

**Algorithm 1** OLSGA (Semi-bandit Feedback)

---
1: **Input:** Convex set $\mathcal{K}$, time horizon $T$
2: Initialize $\mathbf{x}_1 \in \mathcal{K}$, $\lambda_1 = 0$.
3: **for** $t \in [T]$ **do**
4:     Play $\mathbf{x}_t$, obtain $f_t(\mathbf{x}_t)$ and $\widetilde{\nabla} f_t(\mathbf{x}_t)$ and $\boldsymbol{p}_t$
5:     Compute $\widehat{\boldsymbol{p}}_t = \frac{1}{t} \sum_{s=1}^{t} \boldsymbol{p}_s$
6:     Compute

$$\widetilde{\nabla}_x \widehat{\mathcal{L}}_t(\mathbf{x}_t, \lambda_t) = \widetilde{\nabla} f_t(\mathbf{x}_t) - 2\lambda_t \nabla \widetilde{g}_t(\mathbf{x}_t) \tag{5}$$

$$\nabla_\lambda \mathcal{L}_t(\mathbf{x}_t, \lambda_t) = -\widetilde{g}_t(\mathbf{x}_t) + \delta\eta\lambda_t \tag{6}$$

7:     Update $\mathbf{x}_t$ and $\lambda_t$:

$$\mathbf{x}_{t+1} = \Pi_{\mathcal{K}}(\mathbf{x}_t + \eta\widetilde{\nabla}_x \widehat{\mathcal{L}}_t(\mathbf{x}_t, \lambda_t)) \tag{7}$$

$$\lambda_{t+1} = \Pi_{[0,+\infty)}(\lambda_t - \eta\nabla_\lambda \mathcal{L}_t(\mathbf{x}_t, \lambda_t)) \tag{8}$$

8: **end for**

---

**Remark 1.** A notable difference between our algorithm and all prior works addressing online DR-submodular maximization with long-term constraints (e.g., [29, 31]) is that our algorithm can handle search spaces that do not necessarily include $\mathbf{0}$. This distinction bears importance, particularly when considering scenarios where we can only query values within the constraint set. In such cases, $1/2$ has been conjectured to be the optimal approximation ratio ([28] section B in Appendix). We refer to Appendix H for motivation examples where searching over $\mathcal{K} \cup \{\mathbf{0}\}$ is not applicable.

Now, we establish the regret and constraint violation achievable by our proposed Algorithm 1. Before delving into the main theorem, we first present three lemmas, which are adapted from [29] and are essential for achieving high probability bounds. Given the slight difference in the definition of $\widehat{\boldsymbol{p}}$, we provide the proofs in Appendices A to C respectively. First, Lemma 3 demonstrates that with high probability, the empirical estimate $\widehat{\boldsymbol{p}}_t$ is close to its mean $\boldsymbol{p}$.

**Lemma 3.** *The following holds with probability at least $1 - \varepsilon$:*

$$\sum_{t=1}^{T} \|\widehat{\boldsymbol{p}}_t - \boldsymbol{p}\| \leq Q\beta\sqrt{T \log\left(\frac{2nT}{\varepsilon}\right)},$$

*where $Q > 0$ is some universal constant.*

Next, Lemma 4 establishes that with high probability, the $\widehat{g}(\cdot)$ computed using $\widehat{\boldsymbol{p}}_t$ and $g(\cdot)$ calculated using $\boldsymbol{p}$ are close.

**Lemma 4.** *Let $\mathbf{x} \in \mathcal{K}$ be fixed. For a fixed $t \in [T]$ and $\left\{\gamma_t := \sqrt{\frac{2}{t}C^2 \log\left(\frac{2T}{\varepsilon}\right)}\right\}_{t=1}^{T}$, $|\widehat{g}_t(\mathbf{x}) - g(\mathbf{x})| \leq \gamma_t$ holds with probability at least $1 - \frac{\varepsilon}{T}$.*

Finally, Lemma 5 provides an upper bound for the total constraint violation.

**Lemma 5.** *Let $\{\gamma_t\}_{t=1}^{T}$ be defined as in Lemma 4, then the following holds:*

$$C_T \leq \sum_{t=1}^{T} \widetilde{g}_t(\mathbf{x}_t) + r\sum_{t=1}^{T} \|\widehat{\boldsymbol{p}}_t - \boldsymbol{p}\| + \sum_{t=1}^{T} \gamma_t. \tag{9}$$

Armed with these results, we can now establish regret and constraint violation bounds for Algorithm 1.

**Theorem 1.** *Let Assumptions 1 2 3 4 be satisfied. Let $U = \max\{G, C\}$. Choose $\eta = \frac{d}{U\sqrt{T}}$ and $\delta = 8\beta^2$. Let $\mathbf{x}_1, \dots, \mathbf{x}_T$ be the sequence of solutions obtained by Algorithm 1. When $T$ is sufficiently large, i.e., $T \geq \frac{64d^2\beta^2}{U^2}$, we have the following $\frac{1}{2}$-regret and constraint violation bounds with probability at least $1 - \varepsilon$:*

$$\mathbb{E}[R_T] = \mathcal{O}(\sqrt{T}) \text{ and } C_T = \mathcal{O}(T^{3/4}).$$

The complete theorem statement and the complete proof are in Appendix D.

*Partial Proof:* From the update of $\mathbf{x}_t$, we have that for any $\mathbf{x} \in \mathcal{K}$,

$$
\begin{aligned}
\|\mathbf{x}_{t+1} - \mathbf{x}\|^2 &= \|\Pi_\mathcal{K}(\mathbf{x}_t + \eta\widetilde{\nabla}_x\widehat{\mathcal{L}}_t(\mathbf{x}_t, \lambda_t)) - \mathbf{x}\|^2 \\
&\leq \|\mathbf{x}_t + \eta\widetilde{\nabla}_x\widehat{\mathcal{L}}_t(\mathbf{x}_t, \lambda_t) - \mathbf{x}\|^2 \\
&\leq \|\mathbf{x}_t - \mathbf{x}\|^2 + \eta^2\|\widetilde{\nabla}_x\widehat{\mathcal{L}}_t(\mathbf{x}_t, \lambda_t)\|^2 - 2\eta(\mathbf{x} - \mathbf{x}_t)^\top\widetilde{\nabla}_x\widehat{\mathcal{L}}_t(\mathbf{x}_t, \lambda_t). \quad (10)
\end{aligned}
$$

Rearranging, and using Assumption 4 and Assumption 3 we have

$$
(\mathbf{x} - \mathbf{x}_t)^\top\widetilde{\nabla}_x\widehat{\mathcal{L}}_t(\mathbf{x}_t, \lambda_t) \leq \frac{1}{2\eta}(\|\mathbf{x}_t - \mathbf{x}\|^2 - \|\mathbf{x}_{t+1} - \mathbf{x}\|^2) + \eta G^2 + 4\eta\beta^2\lambda_t^2. \quad (11)
$$

Applying similar steps on the $\lambda$ updates, we establish

$$
(\lambda - \lambda_t)^\top\nabla_\lambda\mathcal{L}_t(\mathbf{x}_t, \lambda_t) \geq -\frac{1}{2\eta}(\|\lambda_t - \lambda\|^2 - \|\lambda_{t+1} - \lambda\|^2) - C^2\eta - 2\eta\gamma_t^2 - 2\delta^2\eta^3\lambda_t^2. \quad (12)
$$

From monotonicity and DR-submodularity of $\mathbb{E}[f_t(\mathbf{x})]$, we have

$$
\begin{aligned}
&\mathbb{E}[\mathcal{L}_t(\mathbf{x}, \lambda_t) - 2\mathcal{L}_t(\mathbf{x}_t, \lambda_t)] \\
&= \mathbb{E}[\mathbb{E}[\mathcal{L}_t(\mathbf{x}, \lambda_t) - 2\mathcal{L}_t(\mathbf{x}_t, \lambda_t)|\mathbf{x}_t]] \\
&\leq \mathbb{E}[(\mathbf{x} - \mathbf{x}_t)^\top\nabla_x\mathbb{E}[\widehat{\mathcal{L}}_t(\mathbf{x}_t, \lambda_t)|\mathbf{x}_t]] + \lambda_t\widetilde{g}_t(\mathbf{x}) - \frac{\delta\eta}{2}\lambda_t^2 \quad \text{(Lemma 1)} \\
&\leq \frac{1}{2\eta}\mathbb{E}[\|\mathbf{x}_t - \mathbf{x}\|^2 - \|\mathbf{x}_{t+1} - \mathbf{x}\|^2] + G^2\eta + 4\eta\beta^2\lambda_t^2 + \lambda_t\widetilde{g}_t(\mathbf{x}) - \frac{\delta\eta}{2}\lambda_t^2, \quad (13)
\end{aligned}
$$

where (13) follows from (11). Similarly, from convexity of function $\mathcal{L}_t(\mathbf{x}, \lambda)$ w.r.t $\lambda$, we have

$$
\begin{aligned}
\mathcal{L}_t(\mathbf{x}_t, \lambda) - \mathcal{L}_t(\mathbf{x}_t, \lambda_t) &\geq (\lambda - \lambda_t)^\top\nabla_\lambda\mathcal{L}_t(\mathbf{x}_t, \lambda_t) \\
&\geq -\frac{1}{2\eta}(\|\lambda_t - \lambda\|^2 - \|\lambda_{t+1} - \lambda\|^2) - C^2\eta - 2\eta\gamma_t^2 - 2\delta^2\eta^3\lambda_t^2, \quad (14)
\end{aligned}
$$

where (14) follows from (12). Subtracting two times (14) from (13), and sum $t$ over 1 through $T$, we get

$$
\begin{aligned}
&\sum_{t=1}^T\mathbb{E}[\mathcal{L}_t(\mathbf{x}, \lambda_t) - 2\mathcal{L}_t(\mathbf{x}_t, \lambda)] \\
&\leq \frac{d^2}{2\eta} + \frac{\lambda^2}{\eta} + G^2\eta T + \eta\beta^2\sum_{t=1}^T\lambda_t^2 + 2C^2\eta T + 4\eta\sum_{t=1}^T\gamma_t^2 + 4\delta^2\eta^3\sum_{t=1}^T\lambda_t^2 + \lambda_t\widetilde{g}_t(\mathbf{x}) - \frac{\delta\eta}{2}\lambda_t^2,
\end{aligned}
$$
$$(15)$$

Expanding the left hand side of (15) and rearranging, we deduce

$$
\begin{aligned}
&\sum_{t=1}^T[f(\mathbf{x}) - 2f(\mathbf{x}_t)] + \left[2\lambda\sum_{t=1}^T\widetilde{g}_t(\mathbf{x}_t) - \left(\delta\eta T + \frac{1}{\eta}\right)\lambda^2\right] \\
&\leq 2\sum_{t=1}^T\lambda_t\widetilde{g}_t(\mathbf{x}) + \eta\left(4\beta^2 + 4\delta^2\eta^2 - \delta\right)\sum_{t=1}^T\lambda_t^2 + \frac{d^2}{2\eta} + G^2\eta T + 2C^2\eta T + 4\eta\sum_{t=1}^T\gamma_t^2. \quad (16)
\end{aligned}
$$

To ensure that the equation $4\beta^2 + 4\delta^2\eta^2 - \delta = 0$ has real roots, we require $T \geq \frac{64d^2\beta^2}{U^2}$. Setting $\delta = 8\beta^2$ ensures that $4\beta^2 + 4\delta^2\eta^2 - \delta \leq 0$. Set $\mathbf{x} = \mathbf{x}^*$; From Lemma 4, with probability at least $1 - \frac{\varepsilon}{T}$, $\widetilde{g}_t(\mathbf{x}^*) = \widehat{g}_t(\mathbf{x}^*) - \gamma_t \leq g(\mathbf{x}^*)$ holds. since $\mathbf{x}^*$ satisfies the long term constraint, we have $g(\mathbf{x}^*) \leq 0$. Thus, we can drop the first two terms in the RHS of (16) and by union bound, we get with probability at least $1 - \varepsilon$,

$$
\sum_{t=1}^T[f(\mathbf{x}^*) - 2f(\mathbf{x}_t)] + \left[2\lambda\sum_{t=1}^T\widetilde{g}_t(\mathbf{x}_t) - \left(\delta\eta T + \frac{1}{\eta}\right)\lambda^2\right] \leq \frac{d^2}{2\eta} + G^2\eta T + 2C^2\eta T + 4\eta\sum_{t=1}^T\gamma_t^2.
$$
$$(17)$$

Maximizing the LHS of (17) with respect to $\lambda$ over the range $[0, +\infty)$, we get a solution of $\lambda = \frac{\left[\sum_{t=1}^{T} \widetilde{g}_t(\mathbf{x}_t)\right]_+}{\delta\eta T + 1/\eta}$. Plugging this into (17) gives us

$$\sum_{t=1}^{T}[f(\mathbf{x}^*) - 2f(\mathbf{x}_t)] + \frac{\left[\sum_{t=1}^{T} \widetilde{g}_t(\mathbf{x}_t)\right]_+^2}{\delta\eta T + 1/\eta} \leq \frac{d^2}{2\eta} + G^2\eta T + 2C^2\eta T + 4\eta\sum_{t=1}^{T}\gamma_t^2. \quad (18)$$

Plugging in $U = \max\{G, C\}$ and $\eta = \frac{d}{U\sqrt{T}}$, we have with probability at least $1 - \varepsilon$,

$$\sum_{t=1}^{T}[f(\mathbf{x}^*) - 2f(\mathbf{x}_t)] + \frac{\left[\sum_{t=1}^{T} \widetilde{g}_t(\mathbf{x}_t)\right]_+^2}{\delta\eta T + 1/\eta} \leq \frac{7dU}{2}\sqrt{T} + 8dU\log\frac{2T}{\varepsilon}, \quad (19)$$

where we used the fact that $\sum_{t=1}^{T} \frac{1}{\sqrt{t}} \leq 2\sqrt{T}$. This gives us our result on objective regret:

$$\sum_{t=1}^{T}\left[\frac{1}{2}f(\mathbf{x}^*) - f(\mathbf{x}_t)\right] = \mathcal{O}(\sqrt{T}). \quad (20)$$

The detailed proof, including constraint violation steps, are provided in Appendix D.

## 6    First Order Full-information Case

In this section, we introduce our second proposed algorithm for online DR-submodular maximization subject to stochastic long-term constraints under the first-order full-information feedback. In the interest of space, the algorithm is presented in Algorithm 2 in the Appendix. The overall structure of the algorithm is similar to Algorithm 1, but the primal update uses the gradient of the non-oblivious function:

$$\widetilde{\nabla}_x\widehat{\mathcal{L}}_t(\mathbf{x}_t, \lambda_t) = \widetilde{\nabla}F_t(\mathbf{x}_t) - \lambda_t\nabla\widetilde{g}_t(\mathbf{x}_t), \text{and}$$
$$\mathbf{x}_{t+1} = \Pi_{\mathcal{K}}(\mathbf{x}_t + \eta\widetilde{\nabla}_x\widehat{\mathcal{L}}_t(\mathbf{x}_t, \lambda_t)),$$

where the non-oblivious function $F$ is defined by its gradient: $\nabla F(\mathbf{x}) = \int_0^1 e^{z-1}\nabla f(z \cdot \mathbf{x})dz$. As we discussed in Section 3, the non-oblivious function $F$ plays an important role in obtaining the optimal approximation ratio $1 - 1/e$. However, calculating the gradient of the non-oblivious function $F(\mathbf{x})$ can be challenging, especially when only unbiased estimates of the gradients are available. To overcome this, [42] presents a computational approach for obtaining an unbiased estimate of the gradient of $F(\mathbf{x})$ through sampling (Lines 6 and 7). The following lemma indicates that $(1 - 1/e)\widetilde{\nabla}f_t(z * \mathbf{x})$ is an unbiased estimator of $\nabla F(\mathbf{x})$ with bounded variance.

**Lemma 6.** *If $z$ is sampled from r.v. $\mathbf{Z}$ as in line 6 of Algorithm 2, $\mathbb{E}[\widetilde{\nabla}f_t(\mathbf{x}) \mid \mathbf{x}] = \nabla f(\mathbf{x})$, and $\mathbb{E}\left[\|\widetilde{\nabla}f_t(\mathbf{x}) - \nabla f(\mathbf{x})\|^2 \mid \mathbf{x}\right] \leq \sigma^2$, we have*

*(i)* $\mathbb{E}\left[(1 - 1/e)\widetilde{\nabla}f_t(z * \mathbf{x})\mid \mathbf{x}\right] = \nabla F(\mathbf{x})$;

*(ii)* $\mathbb{E}\left[\left\|(1 - 1/e)\widetilde{\nabla}f_t(z * \mathbf{x}) - \nabla F(\mathbf{x})\right\|^2 \mid \mathbf{x}\right] \leq \sigma_1^2$, *where* $\sigma_1^2 = 2(1 - 1/e)^2\sigma^2 + \frac{2\beta^2\bar{r}^2(1-1/e)}{3}$.

With the unbiased estimator of the gradient of the non-oblivious function, we show the following regret and constraint violation guarantee for our Algorithm 2 in Appendix E:

**Theorem 2.** *Let Assumptions 1 2 3 4 be satisfied. Let $U = \max\{G_F, C\}$. Choosing $\eta = \frac{d}{U\sqrt{T}}$ and $\delta = 4\beta^2$. Let $\mathbf{x}_t, t \in [T]$ be the sequence of solutions obtained by Algorithm 2. When $T$ is sufficiently large, i.e., $T \geq \frac{16d^2\beta^2}{U^2}$, we have the following $(1 - 1/e)$-regret and constraint violation bounds with probability at least $1 - \varepsilon$:*

$$\mathbb{E}[R_T] = \mathcal{O}(\sqrt{T}) \text{ and } C_T = \mathcal{O}(T^{3/4}).$$

## 7 Conclusions

In this paper, we address the problem of stochastic DR-submodular maximization with stochastic long-term constraints over a general convex set. We introduce the first algorithm for this setting, attaining $\mathcal{O}(\sqrt{T})$ regret and $\mathcal{O}(T^{3/4})$ constraint violation bounds. Notably, our algorithm operates in both the semi-bandit feedback and first-order full-information setting, requiring only 1 gradient query per round, while all previous works operate in the full-information setting with $\sqrt{T}$ gradient queries per round. Extension of the results here to upper-linearizable functions in [27] is an open direction.

## 8 Acknowledgement

This work was supported in part by the National Science Foundation under grants CCF-2149588 and CCF-2149617. We acknowledge Yiyang (Roy) Lu for helpful feedback.

## References

[1] S. Agrawal and N. R. Devanur. Bandits with concave rewards and convex knapsacks. *Proceedings of the fifteenth ACM conference on Economics and computation*, 2014.

[2] F. R. Bach. Submodular functions: from discrete to continuous domains. *Mathematical Programming*, 175:419 – 459, 2015.

[3] A. Badanidiyuru, R. D. Kleinberg, and A. Slivkins. Bandits with knapsacks. *2013 IEEE 54th Annual Symposium on Foundations of Computer Science*, pages 207–216, 2013.

[4] S. Balseiro and Y. Gur. Learning in repeated auctions with budgets: Regret minimization and equilibrium. *Management Science*, 65:3952–3968, 09 2019. doi: 10.1287/mnsc.2018.3174.

[5] A. A. Bian, B. Mirzasoleiman, J. Buhmann, and A. Krause. Guaranteed Non-convex Optimization: Submodular Maximization over Continuous Domains. In *AISTATS*, volume 54, pages 111–120. PMLR, 20–22 Apr 2017.

[6] Y. Bian, J. M. Buhmann, and A. Krause. Continuous submodular function maximization. *ArXiv*, abs/2006.13474, 2020.

[7] L. Chen, C. Harshaw, H. Hassani, and A. Karbasi. Projection-free online optimization with stochastic gradient: From convexity to submodularity. In J. Dy and A. Krause, editors, *Proceedings of the 35th International Conference on Machine Learning*, volume 80 of *Proceedings of Machine Learning Research*, pages 814–823. PMLR, 10–15 Jul 2018.

[8] L. Chen, C. Harshaw, H. Hassani, and A. Karbasi. Projection-free online optimization with stochastic gradient: From convexity to submodularity. In J. Dy and A. Krause, editors, *Proceedings of the 35th International Conference on Machine Learning*, volume 80 of *Proceedings of Machine Learning Research*, pages 814–823. PMLR, 10–15 Jul 2018.

[9] L. Chen, H. Hassani, and A. Karbasi. Online continuous submodular maximization. In A. Storkey and F. Perez-Cruz, editors, *Proceedings of the Twenty-First International Conference on Artificial Intelligence and Statistics*, volume 84 of *Proceedings of Machine Learning Research*, pages 1896–1905. PMLR, 09–11 Apr 2018.

[10] J. Feng, R. Yang, Y. Zhang, and Z. Zhang. Online weakly dr-submodular optimization with stochastic long-term constraints. In D.-Z. Du, D. Du, C. Wu, and D. Xu, editors, *Theory and Applications of Models of Computation*, pages 32–42, Cham, 2022. Springer International Publishing. ISBN 978-3-031-20350-3.

[11] C. Guestrin, A. Krause, and A. P. Singh. Near-optimal sensor placements in gaussian processes. *Proceedings of the 22nd international conference on Machine learning*, 2005.

[12] H. Hassani, M. Soltanolkotabi, and A. Karbasi. Gradient methods for submodular maximization. In *Neural Information Processing Systems*, 2017.

[13] E. Hazan. *Introduction to Online Convex Optimization*. Foundations and Trends in Optimization. Now, Boston, 2016. ISBN 978-1-68083-170-2. doi: 10.1561/2400000013.

[14] R. Jenatton, J. Huang, and C. Archambeau. Adaptive algorithms for online convex optimization with long-term constraints. In M. F. Balcan and K. Q. Weinberger, editors, *Proceedings of The 33rd International Conference on Machine Learning*, volume 48 of *Proceedings of Machine Learning Research*, pages 402–411, New York, New York, USA, 20–22 Jun 2016. PMLR.

[15] C. Jin, P. Netrapalli, R. Ge, S. M. Kakade, and M. I. Jordan. A short note on concentration inequalities for random vectors with subgaussian norm, 2019.

[16] D. Kempe, J. M. Kleinberg, and É. Tardos. Maximizing the spread of influence through a social network. In *Knowledge Discovery and Data Mining*, 2003.

[17] A. Krause and D. Golovin. Submodular function maximization. In *Tractability*, 2014.

[18] N. Liakopoulos, A. Destounis, G. Paschos, T. Spyropoulos, and P. Mertikopoulos. Cautious regret minimization: Online optimization with long-term budget constraints. In K. Chaudhuri and R. Salakhutdinov, editors, *Proceedings of the 36th International Conference on Machine Learning*, volume 97 of *Proceedings of Machine Learning Research*, pages 3944–3952. PMLR, 09–15 Jun 2019.

[19] T. Lin, J. Li, and W. Chen. Stochastic online greedy learning with semi-bandit feedbacks. In *Proceedings of the 29th International Conference on Neural Information Processing Systems*, pages 352–360, 2015.

[20] M. Mahdavi, R. Jin, and T. Yang. Trading regret for efficiency: Online convex optimization with long term constraints. *Journal of Machine Learning Research*, 13(81):2503–2528, 2012.

[21] S. Mannor, J. N. Tsitsiklis, and J. Y. Yu. Online learning with sample path constraints. *Journal of Machine Learning Research*, 10:569–590, 2009.

[22] B. Mirzasoleiman, A. Karbasi, R. Sarkar, and A. Krause. Distributed submodular maximization: Identifying representative elements in massive data. In C. Burges, L. Bottou, M. Welling, Z. Ghahramani, and K. Weinberger, editors, *Advances in Neural Information Processing Systems*, volume 26. Curran Associates, Inc., 2013.

[23] B. Mirzasoleiman, A. Badanidiyuru, and A. Karbasi. Fast constrained submodular maximization: Personalized data summarization. In *International Conference on Machine Learning*, 2016.

[24] R. Niazadeh, N. Golrezaei, J. R. Wang, F. Susan, and A. Badanidiyuru. Online learning via offline greedy algorithms: Applications in market design and optimization. In *Proceedings of the 22nd ACM Conference on Economics and Computation*, pages 737–738, 2021.

[25] G. Nie, M. Agarwal, A. K. Umrawal, V. Aggarwal, and C. J. Quinn. An explore-then-commit algorithm for submodular maximization under full-bandit feedback. In *Uncertainty in Artificial Intelligence*, pages 1541–1551. PMLR, 2022.

[26] G. Nie, Y. Y. Nadew, Y. Zhu, V. Aggarwal, and C. J. Quinn. A framework for adapting offline algorithms to solve combinatorial multi-armed bandit problems with bandit feedback. In A. Krause, E. Brunskill, K. Cho, B. Engelhardt, S. Sabato, and J. Scarlett, editors, *Proceedings of the 40th International Conference on Machine Learning*, volume 202 of *Proceedings of Machine Learning Research*, pages 26166–26198. PMLR, 23–29 Jul 2023.

[27] M. Pedramfar and V. Aggarwal. From linear to linearizable optimization: A novel framework with applications to stationary and non-stationary dr-submodular optimization. In *Thirty-eighth Conference on Neural Information Processing Systems*, 2024.

[28] M. Pedramfar, C. J. Quinn, and V. Aggarwal. A unified approach for maximizing continuous DR-submodular functions. In *Thirty-seventh Conference on Neural Information Processing Systems*, 2023.

[29] P. S. Raut, O. Sadeghi, and M. Fazel. Online dr-submodular maximization: Minimizing regret and constraint violation. In *AAAI Conference on Artificial Intelligence*, 2021.

[30] O. Sadeghi and M. Fazel. Online continuous dr-submodular maximization with long-term budget constraints. In S. Chiappa and R. Calandra, editors, *Proceedings of the Twenty Third International Conference on Artificial Intelligence and Statistics*, volume 108 of *Proceedings of Machine Learning Research*, pages 4410–4419. PMLR, 26–28 Aug 2020.

[31] O. Sadeghi, P. Raut, and M. Fazel. A single recipe for online submodular maximization with adversarial or stochastic constraints. In H. Larochelle, M. Ranzato, R. Hadsell, M. Balcan, and H. Lin, editors, *Advances in Neural Information Processing Systems*, volume 33, pages 14712–14723. Curran Associates, Inc., 2020.

[32] T. Soma, N. Kakimura, K. Inaba, and K. ichi Kawarabayashi. Optimal budget allocation: Theoretical guarantee and efficient algorithm. In *International Conference on Machine Learning*, 2014.

[33] M. J. Streeter and D. Golovin. An online algorithm for maximizing submodular functions. In *Neural Information Processing Systems*, 2008.

[34] J. Vondrák. *Submodularity in Combinatorial Optimization*. Phd thesis, Charles University, 2007.

[35] X. Wei, H. Yu, and M. J. Neely. Online primal-dual mirror descent under stochastic constraints. *Proceedings of the ACM on Measurement and Analysis of Computing Systems*, 4:1 – 36, 2019.

[36] L. A. Wolsey. An analysis of the greedy algorithm for the submodular set covering problem. *Combinatorica*, 2:385–393, 1982.

[37] X. Yi, X. Li, T. Yang, L. Xie, T. Chai, and K. Johansson. Regret and cumulative constraint violation analysis for online convex optimization with long term constraints. In M. Meila and T. Zhang, editors, *Proceedings of the 38th International Conference on Machine Learning*, volume 139 of *Proceedings of Machine Learning Research*, pages 11998–12008. PMLR, 18–24 Jul 2021.

[38] H. Yu and M. J. Neely. A low complexity algorithm with $o(\sqrt{T})$ regret and o(1) constraint violations for online convex optimization with long term constraints. *Journal of Machine Learning Research*, 21(1):1–24, 2020.

[39] H. Yu, M. J. Neely, and X. Wei. Online convex optimization with stochastic constraints. In *Neural Information Processing Systems*, 2017.

[40] J. Yuan and A. Lamperski. Online convex optimization for cumulative constraints. In S. Bengio, H. Wallach, H. Larochelle, K. Grauman, N. Cesa-Bianchi, and R. Garnett, editors, *Advances in Neural Information Processing Systems*, volume 31. Curran Associates, Inc., 2018.

[41] M. Zhang, L. Chen, H. Hassani, and A. Karbasi. Online continuous submodular maximization: From full-information to bandit feedback. In *Advances in Neural Information Processing Systems*, volume 32. Curran Associates, Inc., 2019.

[42] Q. Zhang, Z. Deng, Z. Chen, H. Hu, and Y. Yang. Stochastic continuous submodular maximization: Boosting via non-oblivious function. In *International Conference on Machine Learning*, 2022.

[43] M. A. Zinkevich. Online convex programming and generalized infinitesimal gradient ascent. In *International Conference on Machine Learning*, 2003.

# A  Proof of Lemma 3

Before proceeding to the proof, we first show a lemma:

**Lemma 7.** *For all $t \in [T]$, the following holds:*

$$\mathbb{P}\{\|\boldsymbol{p}_t - \boldsymbol{p}\| \geq \zeta\} \leq 2e^{-\frac{\zeta^2}{2\rho^2}},$$

*where $\rho = c\beta$ for some universal constant $c > 0$.*

*Proof.* From Assumption 1, we have $\|\boldsymbol{p}_t\| \leq \beta$. We can apply Lemma 1 of [15] with sub-Gaussian parameter $\rho = c\beta$ for some universal constant $c > 0$ and the result follows immediately. $\square$

Then we proceed to prove Lemma 3. We restate the lemma as follows:

**Lemma 8.** *The following holds with probability at least $1 - \varepsilon$:*

$$\sum_{t=1}^{T} \|\widehat{\boldsymbol{p}}_t - \boldsymbol{p}\| \leq Q\beta\sqrt{T \log\left(\frac{2nT}{\varepsilon}\right)},$$

*where $Q > 0$ is some universal constant.*

*Proof.* From the definition of $\widehat{\boldsymbol{p}}_t$, we have

$$\|\widehat{\boldsymbol{p}}_t - \boldsymbol{p}\| = \frac{1}{t}\|\sum_{s=1}^{t}(\boldsymbol{p}_s - \boldsymbol{p})\|. \tag{21}$$

Combine $\mathbb{E}[\boldsymbol{p}_t - \boldsymbol{p}] = 0$ and Lemma 7, we can apply Corollary 7 of [15] to the random vector $\{\boldsymbol{p}_s - \boldsymbol{p}\}_{s=1}^{t}$ and obtain with probability at least $1 - \frac{\varepsilon}{T}$,

$$\|\sum_{s=1}^{t}(\boldsymbol{p}_s - \boldsymbol{p})\| \leq c'\sqrt{\sum_{s=1}^{t}\rho^2 \log(\frac{2dT}{\varepsilon})} = c'\rho\sqrt{t \log(\frac{2dT}{\varepsilon})}, \tag{22}$$

where $c' > 0$ is some universal constant. Combining (21) and (22) and apply union bound, we have with probability at least $1 - \frac{\varepsilon}{T}$,

$$
\begin{aligned}
\sum_{t=1}^{T} \|\widehat{\boldsymbol{p}}_t - \boldsymbol{p}\| &= \sum_{t=1}^{T} \frac{1}{t}\|\sum_{s=1}^{t}(\boldsymbol{p}_s - \boldsymbol{p})\| \\
&\leq \sum_{t=1}^{T} c'\rho\sqrt{\log(\frac{2dT}{\varepsilon})/t} \\
&\leq \sum_{t=1}^{T} c'\rho\sqrt{T \log(\frac{2dT}{\varepsilon})},
\end{aligned} \tag{23}
$$

where the last inequality follows from $\sum_{t=1}^{T} \frac{1}{\sqrt{t}} \leq 2\sqrt{T}$. We get the desired result be taking $Q = 2cc'$. $\square$

# B  Proof of Lemma 4

We restate the lemma as follows:

**Lemma 9.** *Let $\mathbf{x} \in \mathcal{K}$ be fixed. For a fixed $t \in [T]$ and $\left\{\gamma_t := \sqrt{\frac{2}{t}C^2 \log\left(\frac{2T}{\varepsilon}\right)}\right\}_{t=1}^{T}$, $|\widehat{g}_t(\mathbf{x}) - g(\mathbf{x})| \leq \gamma_t$ holds with probability at least $1 - \frac{\varepsilon}{T}$.*

*Proof.* Recall from the definition that $\widetilde{g}_t(\mathbf{x}) = \langle \widehat{\boldsymbol{p}}_t, \mathbf{x} \rangle - \frac{B_T}{T} - \gamma_t$ and $\widehat{g}_t(\mathbf{x}) := \langle \widehat{\boldsymbol{p}}_t, \mathbf{x} \rangle - \frac{B_T}{T}$. Note that $\mathbb{E}[\widehat{g}_t(\mathbf{x})] = \mathbb{E}[\langle \widehat{\boldsymbol{p}}_t, \mathbf{x} \rangle - \frac{B_T}{T}] = \langle \boldsymbol{p}, \mathbf{x} \rangle - \frac{B_T}{T} = g(\mathbf{x})$. Recall that we have defined $C := \max_{\boldsymbol{p}' \sim \mathcal{D}_p} \max_{\mathbf{x} \in \mathcal{K}} |\langle \boldsymbol{p}', \mathbf{x} \rangle - \frac{B_T}{T}|$. Thus, we have $\widehat{g}_t(\mathbf{x})$ is bounded within interval $[-C, C]$. Apply Hoeffding's inequality on $\widehat{g}_t(\mathbf{x})$, we get

$$\mathbb{P}\left\{|\widehat{g}_t(x) - g(x)| > \gamma_t\right\} \leq 2\exp\left(-\frac{t\gamma_t^2}{2C^2}\right).$$

Substituting the value of $\gamma_t$ in the right hand side, we get that $\mathbb{P}\left\{|\widehat{g}_t(x) - g(x)| > \gamma_t\right\} \leq \frac{\epsilon}{T}$. $\qquad\square$

## C  Proof of Lemma 5

We restate the lemma as follows:

**Lemma 10.** *Let $\{\gamma_t\}_{t=1}^T$ be defined as in Lemma 4, then the following holds:*

$$C_T \leq \sum_{t=1}^{T} \widetilde{g}_t(\mathbf{x}_t) + r\sum_{t=1}^{T} \|\widehat{\boldsymbol{p}}_t - \boldsymbol{p}\| + \sum_{t=1}^{T} \gamma_t. \tag{24}$$

*Proof.* Bounding $\sum_{t=1}^T \widetilde{g}_t(\mathbf{x}_t)$ from below, we obtain:

$$
\begin{aligned}
\sum_{t=1}^{T} \widetilde{g}_t(\mathbf{x}_t) &= \sum_{t=1}^{T} g(\mathbf{x}_t) + \sum_{t=1}^{T} (\widehat{g}_t(\mathbf{x}_t) - g(\mathbf{x}_t)) - \sum_{t=1}^{T} \gamma_t \\
&\geq \sum_{t=1}^{T} g(\mathbf{x}_t) - \sum_{t=1}^{T} |\widehat{g}_t(\mathbf{x}_t) - g(\mathbf{x}_t)| - \sum_{t=1}^{T} \gamma_t \\
&= C_T - \sum_{t=1}^{T} |\langle \widehat{\boldsymbol{p}}_t - \boldsymbol{p}, \mathbf{x}_t \rangle| - \sum_{t=1}^{T} \gamma_t \\
&\geq C_T - \sum_{t=1}^{T} \|\widehat{\boldsymbol{p}}_t - \boldsymbol{p}\| \|\mathbf{x}_t\| - \sum_{t=1}^{T} \gamma_t \\
&\geq C_T - r\sum_{t=1}^{T} \|\widehat{\boldsymbol{p}}_t - \boldsymbol{p}\| - \sum_{t=1}^{T} \gamma_t.
\end{aligned}
$$

Rearranging the above inequality, we obtain the desired result. $\qquad\square$

## D  Proof of Theorem 1

We restate our theorem as follows:

**Theorem 3.** *Let Assumptions 1 2 3 4 be satisfied. Let $U = \max\{G, C\}$. Choose $\eta = \frac{d}{U\sqrt{T}}$ and $\delta = 8\beta^2$. Let $\mathbf{x}_t$, $t \in [T]$ be the sequence of solutions obtained by Algorithm 1. When $T$ is sufficiently large, i.e., $T \geq \frac{64d^2\beta^2}{U^2}$, we have the following $\frac{1}{2}$-regret and constraint violation bounds with probability at least $1 - \varepsilon$:*

$$\mathbb{E}[R_T] \leq \sum_{t=1}^{T} \left[\frac{1}{2}f(\mathbf{x}^*) - f(\mathbf{x}_t)\right] \leq \frac{7dU}{4}\sqrt{T} + 8dU\log\frac{2T}{\varepsilon} = \mathcal{O}(T^{1/2})$$

*and*

$$
\begin{aligned}
C_T \leq &\sqrt{\left(\frac{7dU}{4}\sqrt{T} + 4dU\log\frac{2T}{\varepsilon} + (F_1 + F_2)T\right) \cdot \left(\frac{8\beta^2 d}{U} + \frac{U}{d}\right)\sqrt{T}} \\
&+ rQ\sigma\sqrt{T\log\left(\frac{2nT}{\varepsilon}\right)} + 2\sqrt{2TC^2\log\left(\frac{2T}{\varepsilon}\right)} \\
&= \mathcal{O}(T^{3/4}).
\end{aligned}
\tag{25}
$$

*Proof.* From the update of $\mathbf{x}_t$, we have that for any $\mathbf{x} \in \mathcal{K}$,

$$
\begin{aligned}
\|\mathbf{x}_{t+1} - \mathbf{x}\|^2 &= \|\Pi_\mathcal{K}(\mathbf{x}_t + \eta\widetilde{\nabla}_x\widehat{\mathcal{L}}_t(\mathbf{x}_t, \lambda_t)) - \mathbf{x}\|^2 \\
&\leq \|\mathbf{x}_t + \eta\widetilde{\nabla}_x\widehat{\mathcal{L}}_t(\mathbf{x}_t, \lambda_t) - \mathbf{x}\|^2 \qquad\qquad\qquad \text{(def. of projection)} \\
&= \|\mathbf{x}_t - \mathbf{x}\|^2 + \eta^2\|\widetilde{\nabla}_x\widehat{\mathcal{L}}_t(\mathbf{x}_t, \lambda_t)\|^2 - 2\eta(\mathbf{x} - \mathbf{x}_t)^\top\widetilde{\nabla}_x\widehat{\mathcal{L}}_t(\mathbf{x}_t, \lambda_t). \qquad (26)
\end{aligned}
$$

Rearranging,

$$
\begin{aligned}
(\mathbf{x} - \mathbf{x}_t)^\top\widetilde{\nabla}_x\widehat{\mathcal{L}}_t(\mathbf{x}_t, \lambda_t) &\leq \frac{1}{2\eta}(\|\mathbf{x}_t - \mathbf{x}\|^2 - \|\mathbf{x}_{t+1} - \mathbf{x}\|^2) + \frac{\eta}{2}\|\widetilde{\nabla}_x\widehat{\mathcal{L}}_t(\mathbf{x}_t, \lambda_t)\|^2 \\
&= \frac{1}{2\eta}(\|\mathbf{x}_t - \mathbf{x}\|^2 - \|\mathbf{x}_{t+1} - \mathbf{x}\|^2) + \frac{\eta}{2}\|\widetilde{\nabla}f(\mathbf{x}_t) - 2\lambda_t\nabla\widetilde{g}_t(x_t)\|^2 \\
&\hspace{10cm} \text{(from (5))} \\
&\leq \frac{1}{2\eta}(\|\mathbf{x}_t - \mathbf{x}\|^2 - \|\mathbf{x}_{t+1} - \mathbf{x}\|^2) + \eta\|\widetilde{\nabla}f(\mathbf{x}_t)\|^2 + 4\eta\lambda_t^2\|\nabla\widetilde{g}_t(x_t)\|^2 \\
&\hspace{8cm} (\|a + b\|^2 \leq 2\|a\|^2 + 2\|b\|^2) \\
&\leq \frac{1}{2\eta}(\|\mathbf{x}_t - \mathbf{x}\|^2 - \|\mathbf{x}_{t+1} - \mathbf{x}\|^2) + \eta G^2 + 4\eta\beta^2\lambda_t^2 \qquad\qquad (27)
\end{aligned}
$$

where (27) follows from Assumption 4 and Assumption 3. Taking expectation with respect to $f_t$, we have

$$
\begin{aligned}
\mathbb{E}[\mathcal{L}_t&(\mathbf{x}, \lambda_t) - 2\mathcal{L}_t(\mathbf{x}_t, \lambda_t)] \\
&= \mathbb{E}[\mathbb{E}[\mathcal{L}_t(\mathbf{x}, \lambda_t) - 2\mathcal{L}_t(\mathbf{x}_t, \lambda_t)|\mathbf{x}_t]] \\
&= \mathbb{E}\left[\mathbb{E}\left[f_t(\mathbf{x}) - \lambda_t\widetilde{g}_t(\mathbf{x}) + \frac{\delta\eta}{2}\lambda_t^2 - 2\left(f_t(\mathbf{x}_t) - \lambda_t\widetilde{g}_t(\mathbf{x}_t) + \frac{\delta\eta}{2}\lambda_t^2\right)|\mathbf{x}_t\right]\right] \\
&= \mathbb{E}[\mathbb{E}[f_t(\mathbf{x}) - 2f_t(\mathbf{x}_t) - 2\lambda_t(\widetilde{g}_t(\mathbf{x}) - \widetilde{g}_t(\mathbf{x}_t))|\mathbf{x}_t]] + \lambda_t\widetilde{g}_t(\mathbf{x}) - \frac{\delta\eta}{2}\lambda_t^2 \\
&\leq \mathbb{E}[(\mathbf{x} - \mathbf{x}_t)^\top\nabla_x\mathbb{E}[f_t(\mathbf{x}_t) - 2\lambda_t(\widetilde{g}_t(\mathbf{x}) - \widetilde{g}_t(\mathbf{x}_t))|\mathbf{x}_t]] + \lambda_t\widetilde{g}_t(\mathbf{x}) - \frac{\delta\eta}{2}\lambda_t^2 \\
&\hspace{8cm} \text{(Assumption 2 and Lemma 1)} \\
&= \mathbb{E}[(\mathbf{x} - \mathbf{x}_t)^\top\nabla_x\mathbb{E}[f(\mathbf{x}_t) - 2\lambda_t(\mathbf{x} - \mathbf{x}_t)^\top\nabla_x\widetilde{g}_t(\mathbf{x}_t)|\mathbf{x}_t]] + \lambda_t\widetilde{g}_t(\mathbf{x}) - \frac{\delta\eta}{2}\lambda_t^2 \\
&\hspace{5cm} (\widetilde{g}_t(\cdot) \text{ is linear so } \widetilde{g}_t(\mathbf{y}) - \widetilde{g}_t(\mathbf{x}) \geq \langle\nabla\widetilde{g}_t(\mathbf{x}), \mathbf{y} - \mathbf{x}\rangle) \\
&\leq \mathbb{E}[\mathbb{E}[(\mathbf{x} - \mathbf{x}_t)^\top\nabla_x f(\mathbf{x}_t) - 2\lambda_t(\mathbf{x} - \mathbf{x}_t)^\top\nabla_x\widetilde{g}_t(\mathbf{x}_t)|\mathbf{x}_t]] + \lambda_t\widetilde{g}_t(\mathbf{x}) - \frac{\delta\eta}{2}\lambda_t^2 \\
&\hspace{8cm} \text{(linearity of expectation)} \\
&= \mathbb{E}[\mathbb{E}[(\mathbf{x} - \mathbf{x}_t)^\top\nabla_x\widehat{\mathcal{L}}_t(\mathbf{x}_t, \lambda_t)|\mathbf{x}_t]] + \lambda_t\widetilde{g}_t(\mathbf{x}) - \frac{\delta\eta}{2}\lambda_t^2 \\
&\leq \frac{1}{2\eta}\mathbb{E}[\|\mathbf{x}_t - \mathbf{x}\|^2 - \|\mathbf{x}_{t+1} - \mathbf{x}\|^2] + G^2\eta + 4\eta\beta^2\lambda_t^2 + \lambda_t\widetilde{g}_t(\mathbf{x}) - \frac{\delta\eta}{2}\lambda_t^2, \qquad (28)
\end{aligned}
$$

where (28) follows from (27). From the update of $\lambda_t$ (8), we have

$$
\begin{aligned}
\|\lambda_{t+1} - \lambda\|^2 &= \|\Pi_{[0,+\infty)}(\lambda_t - \eta\nabla_\lambda\mathcal{L}_t(\mathbf{x}_t, \lambda_t)) - \lambda\|^2 \qquad\qquad\qquad\qquad \text{(from (8))} \\
&\leq \|\lambda_t - \eta\nabla_\lambda\mathcal{L}_t(\mathbf{x}_t, \lambda_t) - \lambda\|^2 \qquad\qquad\qquad\qquad\quad \text{(def. of projection)} \\
&= \|\lambda - \lambda_t\|^2 + \eta^2\|\nabla_\lambda\mathcal{L}_t(\mathbf{x}_t, \lambda_t)\|^2 + 2\eta(\lambda - \lambda_t)^\top\nabla_\lambda\mathcal{L}_t(\mathbf{x}_t, \lambda_t). \qquad (29)
\end{aligned}
$$

Rearranging,

$$(\lambda - \lambda_t)^\top \nabla_\lambda \mathcal{L}_t(\mathbf{x}_t, \lambda_t) \geq -\frac{1}{2\eta}(\|\lambda_t - \lambda\|^2 - \|\lambda_{t+1} - \lambda\|^2) - \frac{\eta}{2}\|\nabla_\lambda \mathcal{L}_t(\mathbf{x}_t, \lambda_t)\|^2$$

$$= -\frac{1}{2\eta}(\|\lambda_t - \lambda\|^2 - \|\lambda_{t+1} - \lambda\|^2) - \frac{\eta}{2}\| - \widetilde{g}_t(\mathbf{x}_t) + \delta\eta\lambda_t\|^2$$

<div align="right">(from (6))</div>

$$= -\frac{1}{2\eta}(\|\lambda_t - \lambda\|^2 - \|\lambda_{t+1} - \lambda\|^2) - \frac{\eta}{2}\| - \widehat{g}_t(\mathbf{x}_t) - \gamma_t + \delta\eta\lambda_t\|^2$$

<div align="right">(by def. $\widehat{g}_t$)</div>

$$\geq -\frac{1}{2\eta}(\|\lambda_t - \lambda\|^2 - \|\lambda_{t+1} - \lambda\|^2) - \eta\|\widehat{g}_t(\mathbf{x}_t)\|^2 - 2\eta\gamma_t^2 - 2\delta^2\eta^3\lambda_t^2$$

<div align="right">(apply $\|a+b\|^2 \leq 2\|a\|^2 + 2\|b\|^2$ twice)</div>

$$\geq -\frac{1}{2\eta}(\|\lambda_t - \lambda\|^2 - \|\lambda_{t+1} - \lambda\|^2) - C^2\eta - 2\eta\gamma_t^2 - 2\delta^2\eta^3\lambda_t^2, \quad (30)$$

where the last inequality follows from the definition of $C := \max_{\mathbf{p}' \sim \mathcal{D}_p} \max_{\mathbf{x} \in \mathcal{K}} |\langle \mathbf{p}', \mathbf{x}\rangle - \frac{B_T}{T}|$ and $\widehat{g}_t(\mathbf{x}) := \langle \widehat{\mathbf{p}}_t, \mathbf{x}\rangle - \frac{B_T}{T}$. From convexity of function $\mathcal{L}_t(\mathbf{x}, \lambda)$ w.r.t $\lambda$, we have

$$\mathcal{L}_t(\mathbf{x}_t, \lambda) - \mathcal{L}_t(\mathbf{x}_t, \lambda_t) \geq (\lambda - \lambda_t)^\top \nabla_\lambda \mathcal{L}_t(\mathbf{x}_t, \lambda_t)$$

$$\geq -\frac{1}{2\eta}(\|\lambda_t - \lambda\|^2 - \|\lambda_{t+1} - \lambda\|^2) - C^2\eta - 2\eta\gamma_t^2 - 2\delta^2\eta^3\lambda_t^2 \quad (31)$$

where (31) follows from (30). Subtracting two times (31) from (28), we get

$$\mathbb{E}[\mathcal{L}_t(\mathbf{x}, \lambda_t) - 2\mathcal{L}_t(\mathbf{x}_t, \lambda)] \leq \frac{1}{2\eta}\mathbb{E}[\|\mathbf{x}_t - \mathbf{x}\|^2 - \|\mathbf{x}_{t+1} - \mathbf{x}\|^2] + \frac{1}{\eta}(\|\lambda_t - \lambda\|^2 - \|\lambda_{t+1} - \lambda\|^2)$$

$$+ G^2\eta + 4\eta\beta^2\lambda_t^2 + 2C^2\eta + 4\eta\gamma_t^2 + 4\delta^2\eta^3\lambda_t^2 + \lambda_t\widetilde{g}_t(\mathbf{x}) - \frac{\delta\eta}{2}\lambda_t^2. \quad (32)$$

Summing (32) for $t \in [T]$, we have

$$\sum_{t=1}^{T}\mathbb{E}[\mathcal{L}_t(\mathbf{x}, \lambda_t) - 2\mathcal{L}_t(\mathbf{x}_t, \lambda)]$$

$$\leq \frac{1}{2\eta}\sum_{t=1}^{T}\mathbb{E}[\|\mathbf{x}_t - \mathbf{x}\|^2 - \|\mathbf{x}_{t+1} - \mathbf{x}\|^2] + \frac{1}{\eta}\sum_{t=1}^{T}(\|\lambda_t - \lambda\|^2 - \|\lambda_{t+1} - \lambda\|^2)$$

$$+ G^2\eta T + 4\eta\beta^2\sum_{t=1}^{T}\lambda_t^2 + 2C^2\eta T + 4\eta\sum_{t=1}^{T}\gamma_t^2 + 4\delta^2\eta^3\sum_{t=1}^{T}\lambda_t^2 + \sum_{t=1}^{T}\lambda_t\widetilde{g}_t(\mathbf{x}) - \frac{\delta\eta}{2}\sum_{t=1}^{T}\lambda_t^2$$

$$\leq \frac{1}{2\eta}\mathbb{E}[\|\mathbf{x}_1 - \mathbf{x}\|^2 - \|\mathbf{x}_{T+1} - \mathbf{x}\|^2] + \frac{1}{\eta}(\|\lambda_1 - \lambda\|^2 - \|\lambda_{T+1} - \lambda\|^2)$$

$$+ G^2\eta T + 4\eta\beta^2\sum_{t=1}^{T}\lambda_t^2 + 2C^2\eta T + 4\eta\sum_{t=1}^{T}\gamma_t^2 + 4\delta^2\eta^3\sum_{t=1}^{T}\lambda_t^2 + \sum_{t=1}^{T}\lambda_t\widetilde{g}_t(\mathbf{x}) - \frac{\delta\eta}{2}\sum_{t=1}^{T}\lambda_t^2$$

<div align="right">(telescoping series)</div>

$$\leq \frac{1}{2\eta}\mathbb{E}[\|\mathbf{x}_1 - \mathbf{x}_{T+1}\|^2] + \frac{1}{\eta}(\|\lambda_1 - \lambda\|^2 - \|\lambda_{T+1} - \lambda\|^2)$$

$$+ G^2\eta T + 4\eta\beta^2\sum_{t=1}^{T}\lambda_t^2 + 2C^2\eta T + 4\eta\sum_{t=1}^{T}\gamma_t^2 + 4\delta^2\eta^3\sum_{t=1}^{T}\lambda_t^2 + \sum_{t=1}^{T}\lambda_t\widetilde{g}_t(\mathbf{x}) - \frac{\delta\eta}{2}\sum_{t=1}^{T}\lambda_t^2$$

<div align="right">(triangle inequality)</div>

$$\leq \frac{d^2}{2\eta} + \frac{\lambda^2}{\eta} + G^2\eta T + \eta\beta^2\sum_{t=1}^{T}\lambda_t^2 + 2C^2\eta T + 4\eta\sum_{t=1}^{T}\gamma_t^2 + 4\delta^2\eta^3\sum_{t=1}^{T}\lambda_t^2 + \sum_{t=1}^{T}\lambda_t\widetilde{g}_t(\mathbf{x}) - \frac{\delta\eta}{2}\sum_{t=1}^{T}\lambda_t^2,$$

<div align="right">(33)</div>

where (33) uses Assumption 1 with def. of diameter $d$, $\lambda_1 = 0$, and $-\|\cdot\|^2 \le 0$. Expanding the left hand side of (33), we deduce

$$\sum_{t=1}^{T}[f(\mathbf{x}) - 2f(\mathbf{x}_t)] + \sum_{t=1}^{T}[2\lambda\widetilde{g}_t(\mathbf{x}_t) - \lambda_t\widetilde{g}_t(\mathbf{x})] + \sum_{t=1}^{T}\left[\frac{\delta\eta\lambda_t^2}{2} - \delta\eta\lambda^2\right]$$

$$\le \frac{d^2}{2\eta} + \frac{\lambda^2}{\eta} + G^2\eta T + 4\eta\beta^2\sum_{t=1}^{T}\lambda_t^2 + 2C^2\eta T + 4\eta\sum_{t=1}^{T}\gamma_t^2 + 4\delta^2\eta^3\sum_{t=1}^{T}\lambda_t^2 + \sum_{t=1}^{T}\lambda_t\widetilde{g}_t(\mathbf{x}) - \frac{\delta\eta}{2}\sum_{t=1}^{T}\lambda_t^2.$$

$$(34)$$

Rearranging, we have

$$\sum_{t=1}^{T}[f(\mathbf{x}) - 2f(\mathbf{x}_t)] + \left\{2\lambda\sum_{t=1}^{T}\widetilde{g}_t(\mathbf{x}_t) - \left(\delta\eta T + \frac{1}{\eta}\right)\lambda^2\right\}$$

$$\le 2\sum_{t=1}^{T}\lambda_t\widetilde{g}_t(\mathbf{x}) + \eta\left(4\beta^2 + 4\delta^2\eta^2 - \delta\right)\sum_{t=1}^{T}\lambda_t^2 + \frac{d^2}{2\eta} + G^2\eta T + 2C^2\eta T + 4\eta\sum_{t=1}^{T}\gamma_t^2. \quad (35)$$

Set $\mathbf{x} = \mathbf{x}^*$. From Lemma 4, with probability at least $1 - \frac{\varepsilon}{T}$, $\widetilde{g}_t(\mathbf{x}^*) = \widehat{g}_t(\mathbf{x}^*) - \gamma_t \le g(\mathbf{x}^*)$ holds. Since $\mathbf{x}^*$ satisfies the long term constraint, we have $g(\mathbf{x}^*) \le 0$. By the union bound, we get with probability at least $1 - \varepsilon$ that for the first term on the RHS of (35)

$$2\sum_{t=1}^{T}\lambda_t\widetilde{g}_t(\mathbf{x}^*) \le 2g(\mathbf{x}^*)\sum_{t=1}^{T}\lambda_t \le 0.$$

Now we choose $\delta$ such that $4\beta^2 + 4\delta^2\eta^2 - \delta \le 0$ (so that the second term on the RHS of (35) will be negative and can be dropped with an upper bound). This is a quadratic term w.r.t. $\delta$. It is easy to verify that the quadratic formula has real roots when $\eta \le \frac{1}{8\beta}$. As we choose $\eta = \frac{d}{U\sqrt{T}}$, this will give us the condition that $T$ needs to be sufficiently large, i.e., $T \ge \frac{64d^2\beta^2}{U^2}$. We can simply choose $\delta = 8\beta^2$.

Applying both of those inequalities to (35) and by union bound, we get with probability at least $1 - \varepsilon$,

$$\sum_{t=1}^{T}[f(\mathbf{x}^*) - 2f(\mathbf{x}_t)] + \left\{2\lambda\sum_{t=1}^{T}\widetilde{g}_t(\mathbf{x}_t) - (\delta\eta T + \frac{1}{\eta})\lambda^2\right\}$$

$$\le \frac{d^2}{2\eta} + G^2\eta T + 2C^2\eta T + 4\eta\sum_{t=1}^{T}\gamma_t^2. \quad (36)$$

Maximizing the LHS of (36) with respect to $\lambda$, over the range $[0, +\infty)$, we get a solution of $\lambda = \frac{\left[\sum_{t=1}^{T}\widetilde{g}_t(\mathbf{x}_t)\right]_+}{\delta\eta T + 1/\eta}$. Plugging this into (36) gives us

$$\sum_{t=1}^{T}[f(\mathbf{x}^*) - 2f(\mathbf{x}_t)] + \frac{\left[\sum_{t=1}^{T}\widetilde{g}_t(\mathbf{x}_t)\right]_+^2}{\delta\eta T + 1/\eta} \le \frac{d^2}{2\eta} + G^2\eta T + 2C^2\eta T + 4\eta\sum_{t=1}^{T}\gamma_t^2. \quad (37)$$

Let $U = \max\{G, C\}$. Choosing $\eta = \frac{d}{U\sqrt{T}}$, we have with probability at least $1 - \varepsilon$,

$$\sum_{t=1}^{T} [f(\mathbf{x}^*) - 2f(\mathbf{x}_t)] + \frac{\left[\sum_{t=1}^{T} \widetilde{g}_t(\mathbf{x}_t)\right]_+^2}{\delta\eta T + 1/\eta}$$

$$\leq \frac{d\max\{G, C\}}{2}\sqrt{T} + \frac{G^2 d}{\max\{G, C\}}\sqrt{T} + \frac{2C^2 d}{\max\{G, C\}}\sqrt{T} + \frac{8\max\{G, C\}d\log\frac{2T}{\varepsilon}}{\sqrt{T}}\sum_{t=1}^{T}\frac{1}{t}$$

$$\leq \frac{d\max\{G, C\}}{2}\sqrt{T} + \max\{G, C\}d\sqrt{T} + 2\max\{G, C\}d\sqrt{T} + \frac{8Ud\log\frac{2T}{\varepsilon}}{\sqrt{T}}\sum_{t=1}^{T}\frac{1}{t} \tag{38}$$

$$\leq \frac{7dU}{2}\sqrt{T} + \frac{8Ud\log\frac{2T}{\varepsilon}}{\sqrt{T}}\sum_{t=1}^{T}\frac{1}{\sqrt{t}} \tag{39}$$

$$\leq \frac{7dU}{2}\sqrt{T} + 16Ud\log\frac{2T}{\varepsilon}. \tag{40}$$

where Equation (40) uses $\sum_{t=1}^{T}\frac{1}{\sqrt{t}} \leq 2\sqrt{T}$. This gives us

$$\sum_{t=1}^{T}\left[\frac{1}{2}f(\mathbf{x}^*) - f(\mathbf{x}_t)\right] \leq \frac{7dU}{4}\sqrt{T} + 8dU\log\frac{2T}{\varepsilon} = \mathcal{O}(T^{1/2}). \tag{41}$$

Next, we establish our constraint violation bound. Since $F_1 := \max_{\mathbf{x}\in\mathcal{K}}|f(\mathbf{x})|$ and $F_2 := \max_{\mathbf{x},\mathbf{y}\in\mathcal{K}}|f(\mathbf{x}) - f(\mathbf{y})|$, we have

$$|f(\mathbf{x}^*) - 2f(\mathbf{x}_t)| \leq |f(\mathbf{x}^*) - f(\mathbf{x}_t)| + |f(\mathbf{x}_t)| \leq F_1 + F_2, \tag{42}$$

thus, $\sum_{t=1}^{T}[f(\mathbf{x}^*) - 2f(\mathbf{x}_t)] \geq -(F_1 + F_2)T$. Plugging back in (40), we have

$$\frac{\left[\sum_{t=1}^{T}\widetilde{g}_t(\mathbf{x}_t)\right]_+^2}{\delta\eta T + 1/\eta} \leq \frac{7dU}{4}\sqrt{T} + 8dU\log\frac{2T}{\varepsilon} + (F_1 + F_2)T. \tag{43}$$

Rearranging and plug in the value of $\eta$, we have

$$\left[\sum_{t=1}^{T}\widetilde{g}(\mathbf{x}_t)\right]_+ \leq \sqrt{\left(\frac{7dU}{4}\sqrt{T} + 8dU\log\frac{2T}{\varepsilon} + (F_1 + F_2)T\right)\cdot\left(\frac{8\beta^2 d}{U} + \frac{U}{d}\right)\sqrt{T}}. \tag{44}$$

Combining (44) with Lemma 5, we get

$$C_T \leq \sqrt{\left(\frac{7dU}{4}\sqrt{T} + 8dU\log\frac{2T}{\varepsilon} + (F_1 + F_2)T\right)\cdot\left(\frac{8\beta^2 d}{U} + \frac{U}{d}\right)\sqrt{T}} + r\sum_{t=1}^{T}\|\widehat{p}_t - p\| + \sum_{t=1}^{T}\gamma_t$$

$$\leq \sqrt{\left(\frac{7dU}{4}\sqrt{T} + 8dU\log\frac{2T}{\varepsilon} + (F_1 + F_2)T\right)\cdot\left(\frac{8\beta^2 d}{U} + \frac{U}{d}\right)\sqrt{T}}$$

$$\qquad + rQ\sigma\sqrt{T\log\left(\frac{2nT}{\varepsilon}\right)} + \sum_{t=1}^{T}\gamma_t \qquad\qquad \text{(using Lemma 3)}$$

$$\leq \sqrt{\left(\frac{7dU}{4}\sqrt{T} + 8dU\log\frac{2T}{\varepsilon} + (F_1 + F_2)T\right)\cdot\left(\frac{8\beta^2 d}{U} + \frac{U}{d}\right)\sqrt{T}}$$

$$\qquad + rQ\sigma\sqrt{T\log\left(\frac{2nT}{\varepsilon}\right)} + 2\sqrt{2TC^2\log\left(\frac{2T}{\varepsilon}\right)}$$

$$= \mathcal{O}(T^{3/4}).$$

where in the last inequality we plug in the definition of $\gamma_t$. This concludes the proof. $\qquad\square$

**Algorithm 2** OLSGA with First Order Full Information
---
1: **Input:** Convex set $\mathcal{K}$, time horizon $T$
2: Initialize $\mathbf{x}_1 \in \mathcal{K}$, $\lambda_1 = 0$.
3: **for** $t \in [T]$ **do**
4:     Play $\mathbf{x}_t$, obtain $f_t(\mathbf{x}_t)$ and $\widetilde{\nabla} f_t(\cdot)$ and $\boldsymbol{p}_t$
5:     Compute $\widehat{\boldsymbol{p}}_t = \frac{1}{t} \sum_{s=1}^{t} \boldsymbol{p}_s$
6:     Sample $z_t$ from $\mathbf{Z}$ where $\mathrm{P}(\mathbf{Z} \leq z) = \int_0^z \frac{e^{u-1}}{1-e^{-1}} \mathrm{d}u$.
7:     Compute $\widetilde{\nabla} F_t(\mathbf{x}_t) = (1 - 1/e)\widetilde{\nabla} f_t(z_t * \mathbf{x}_t)$
8:     Compute

$$\widetilde{\nabla}_x \widehat{\mathcal{L}}_t(\mathbf{x}_t, \lambda_t) = \widetilde{\nabla} F_t(\mathbf{x}_t) - \lambda_t \nabla \widetilde{g}_t(\mathbf{x}_t) \tag{45}$$

$$\nabla_\lambda \mathcal{L}_t(\mathbf{x}_t, \lambda_t) = -\widetilde{g}_t(\mathbf{x}_t) + \delta\eta\lambda_t \tag{46}$$

9:     Update $\mathbf{x}_t$ and $\lambda_t$:

$$\mathbf{x}_{t+1} = \Pi_{\mathcal{K}}(\mathbf{x}_t + \eta\widetilde{\nabla}_x \widehat{\mathcal{L}}_t(\mathbf{x}_t, \lambda_t)) \tag{47}$$

$$\lambda_{t+1} = \Pi_{[0,+\infty)}(\lambda_t - \eta\nabla_\lambda \mathcal{L}_t(\mathbf{x}_t, \lambda_t)) \tag{48}$$

10: **end for**
---

# E    Proof of Theorem 2

We restate our theorem as follows:

**Theorem 4.** *Let Assumptions 1 2 3 4 be satisfied. Let $U = \max\{G_F, C\}$. Choosing $\eta = \frac{d}{U\sqrt{T}}$ and $\delta = 4\beta^2$. Let $\mathbf{x}_t$, $t \in [T]$ be the sequence of solutions obtained by Algorithm 2. When $T$ is sufficiently large, i.e., $T \geq \frac{32d^2\beta^2}{U^2}$, we have the following $(1 - 1/e)$-regret and constraint violation bounds with probability at least $1 - \varepsilon$:*

$$\mathbb{E}[R_T] \leq \sum_{t=1}^{T}[(1-1/e)f(\mathbf{x}^*) - f(\mathbf{x}_t)] \leq \frac{5dU}{2}\sqrt{T} + 8dU\log\frac{2T}{\varepsilon} = \mathcal{O}(T^{1/2})$$

*and*

$$C_T \leq \sqrt{\left(\frac{5dU}{2}\sqrt{T} + 8dU\log\frac{2T}{\varepsilon} + (\frac{F_1}{e} + F_2)T\right) \cdot \left(\frac{4\beta^2 d}{U} + \frac{U}{d}\right)\sqrt{T}}$$

$$+ rQ\sigma\sqrt{T\log\left(\frac{2nT}{\varepsilon}\right)} + 2\sqrt{2TC^2\log\left(\frac{2T}{\varepsilon}\right)}$$

$$= \mathcal{O}(T^{3/4}).$$

*Proof.* From the update of $\mathbf{x}_t$, we have that for any $\mathbf{x} \in \mathcal{K}$,

$$\begin{aligned}
\|\mathbf{x}_{t+1} - \mathbf{x}\|^2 &= \|\Pi_{\mathcal{K}}(\mathbf{x}_t + \eta\widetilde{\nabla}_x\widehat{\mathcal{L}}_t(\mathbf{x}_t, \lambda_t)) - \mathbf{x}\|^2 \\
&\leq \|\mathbf{x}_t + \eta\widetilde{\nabla}_x\widehat{\mathcal{L}}_t(\mathbf{x}_t, \lambda_t) - \mathbf{x}\|^2 \qquad\qquad \text{(def. of projection)} \\
&= \|\mathbf{x}_t - \mathbf{x}\|^2 + \eta^2\|\widetilde{\nabla}_x\widehat{\mathcal{L}}_t(\mathbf{x}_t, \lambda_t)\|^2 - 2\eta(\mathbf{x} - \mathbf{x}_t)^\top\widetilde{\nabla}_x\widehat{\mathcal{L}}_t(\mathbf{x}_t, \lambda_t). \qquad (49)
\end{aligned}$$

Rearranging,

$$
\begin{aligned}
(\mathbf{x} - \mathbf{x}_t)^\top \widetilde{\nabla}_x \widehat{\mathcal{L}}_t(\mathbf{x}_t, \lambda_t) &\leq \frac{1}{2\eta}(\|\mathbf{x}_t - \mathbf{x}\|^2 - \|\mathbf{x}_{t+1} - \mathbf{x}\|^2) + \frac{\eta}{2}\|\widetilde{\nabla}_x \widehat{\mathcal{L}}_t(\mathbf{x}_t, \lambda_t)\|^2 \\
&= \frac{1}{2\eta}(\|\mathbf{x}_t - \mathbf{x}\|^2 - \|\mathbf{x}_{t+1} - \mathbf{x}\|^2) + \frac{\eta}{2}\|\widetilde{\nabla} F_t(\mathbf{x}_t) - \lambda_t \nabla \widetilde{g}_t(x_t)\|^2 \\
&\hspace{5cm} \text{(from (45))} \\
&\leq \frac{1}{2\eta}(\|\mathbf{x}_t - \mathbf{x}\|^2 - \|\mathbf{x}_{t+1} - \mathbf{x}\|^2) + \eta\|\widetilde{\nabla} F_t(\mathbf{x}_t)\|^2 + \eta\lambda_t^2\|\nabla \widetilde{g}_t(x_t)\|^2 \\
&\hspace{4cm} (\|a + b\|^2 \leq 2\|a\|^2 + 2\|b\|^2) \\
&\leq \frac{1}{2\eta}(\|\mathbf{x}_t - \mathbf{x}\|^2 - \|\mathbf{x}_{t+1} - \mathbf{x}\|^2) + \eta G_F^2 + \eta\beta^2\lambda_t^2 \hspace{1cm} (50)
\end{aligned}
$$

where (50) follows from Assumption 4 and Assumption 3. When $\lambda$ is fixed, we have (taking expectation over $f_t$)

$$
\begin{aligned}
&\mathbb{E}[(1 - 1/e)\mathcal{L}_t(\mathbf{x}, \lambda_t) - \mathcal{L}_t(\mathbf{x}_t, \lambda_t)] \\
&= \mathbb{E}[\mathbb{E}[(1 - 1/e)\mathcal{L}_t(\mathbf{x}, \lambda_t) - \mathcal{L}_t(\mathbf{x}_t, \lambda_t)|\mathbf{x}_t]] \\
&= \mathbb{E}[\mathbb{E}[(1 - 1/e)(f_t(\mathbf{x}) - \lambda_t\widetilde{g}_t(\mathbf{x}) + \frac{\delta\eta}{2}\lambda_t^2) - (f_t(\mathbf{x}_t) - \lambda_t\widetilde{g}_t(\mathbf{x}_t) + \frac{\delta\eta}{2}\lambda_t^2)|\mathbf{x}_t]] \\
&= \mathbb{E}[\mathbb{E}[(1 - 1/e)f_t(\mathbf{x}) - f_t(\mathbf{x}_t) - \lambda_t(\widetilde{g}_t(\mathbf{x}) - \widetilde{g}_t(\mathbf{x}_t))|\mathbf{x}_t]] + \frac{1}{e}\lambda_t\widetilde{g}_t(\mathbf{x}) - \frac{\delta\eta}{2e}\lambda_t^2) \\
&\leq \mathbb{E}[(\mathbf{x} - \mathbf{x}_t)^\top \nabla_x \mathbb{E}[F_t(\mathbf{x}_t) - \lambda_t(\widetilde{g}_t(\mathbf{x}) - \widetilde{g}_t(\mathbf{x}_t))|\mathbf{x}_t]] + \frac{1}{e}\lambda_t\widetilde{g}_t(\mathbf{x}) - \frac{\delta\eta}{2e}\lambda_t^2 \hspace{0.5cm} \text{(Lemma 2)} \\
&= \mathbb{E}[\mathbb{E}[(\mathbf{x} - \mathbf{x}_t)^\top \nabla_x F_t(\mathbf{x}_t) - \lambda_t(\widetilde{g}_t(\mathbf{x}) - \widetilde{g}_t(\mathbf{x}_t))|\mathbf{x}_t]] + \frac{1}{e}\lambda_t\widetilde{g}_t(\mathbf{x}) - \frac{\delta\eta}{2e}\lambda_t^2 \\
&\hspace{7cm} \text{(linearity of expectation)} \\
&\leq \mathbb{E}[\mathbb{E}[(\mathbf{x} - \mathbf{x}_t)^\top \nabla_x F_t(\mathbf{x}_t) - \lambda_t((\mathbf{x} - \mathbf{x}_t)^\top \nabla \widetilde{g}_t(\mathbf{x}_t))|\mathbf{x}_t]] + \frac{1}{e}\lambda_t\widetilde{g}_t(\mathbf{x}) - \frac{\delta\eta}{2e}\lambda_t^2 \\
&\hspace{3cm} (\widetilde{g}_t(\cdot) \text{ is convex so } \widetilde{g}_t(\mathbf{y}) - \widetilde{g}_t(\mathbf{x}) \geq \langle \nabla \widetilde{g}_t(\mathbf{x}), \mathbf{y} - \mathbf{x} \rangle) \\
&= \mathbb{E}[\mathbb{E}[(\mathbf{x} - \mathbf{x}_t)^\top \nabla_x (F_t(\mathbf{x}_t) - \lambda_t\widetilde{g}_t(\mathbf{x}_t))|\mathbf{x}_t]] + \frac{1}{e}\lambda_t\widetilde{g}_t(\mathbf{x}) - \frac{\delta\eta}{2e}\lambda_t^2 \\
&= \mathbb{E}[\mathbb{E}[(\mathbf{x} - \mathbf{x}_t)^\top \nabla_x (F_t(\mathbf{x}_t) - \lambda_t\widetilde{g}_t(\mathbf{x}_t) + \frac{\delta\eta}{2}\lambda_t^2)|\mathbf{x}_t]] + \frac{1}{e}\lambda_t\widetilde{g}_t(\mathbf{x}) - \frac{\delta\eta}{2e}\lambda_t^2 \\
&= \mathbb{E}[\mathbb{E}[(\mathbf{x} - \mathbf{x}_t)^\top \widetilde{\nabla}_x \widehat{\mathcal{L}}_t(\mathbf{x}_t, \lambda_t)|\mathbf{x}_t]] + \frac{1}{e}\lambda_t\widetilde{g}_t(\mathbf{x}) - \frac{\delta\eta}{2e}\lambda_t^2 \\
&\leq \frac{1}{2\eta}\mathbb{E}[\|\mathbf{x}_t - \mathbf{x}\|^2 - \|\mathbf{x}_{t+1} - \mathbf{x}\|^2] + G_F^2\eta + \eta\beta^2\lambda_t^2 + \frac{1}{e}\lambda_t\widetilde{g}_t(\mathbf{x}) - \frac{\delta\eta}{2e}\lambda_t^2, \hspace{0.5cm} (51)
\end{aligned}
$$

where (51) follows from (50). From the update (48) of $\lambda_t$, we have

$$
\begin{aligned}
\|\lambda_{t+1} - \lambda\|^2 &= \|\Pi_{[0,+\infty)}(\lambda_t - \eta\nabla_\lambda\mathcal{L}_t(\mathbf{x}_t, \lambda_t)) - \lambda\|^2 \hspace{1cm} \text{(from (48))} \\
&\leq \|\lambda_t - \eta\nabla_\lambda\mathcal{L}_t(\mathbf{x}_t, \lambda_t) - \lambda\|^2 \hspace{2cm} \text{(def. of projection)} \\
&\leq \|\lambda_t - \lambda\|^2 + \eta^2\|\nabla_\lambda\mathcal{L}_t(\mathbf{x}_t, \lambda_t)\|^2 + 2\eta(\lambda - \lambda_t)^\top\nabla_\lambda\mathcal{L}_t(\mathbf{x}_t, \lambda_t), \hspace{0.5cm} (52)
\end{aligned}
$$

where (52) multiplies through. Rearranging,

$$(\lambda - \lambda_t)^\top \nabla_\lambda \mathcal{L}_t(\mathbf{x}_t, \lambda_t) \geq -\frac{1}{2\eta}(\|\lambda_t - \lambda\|^2 - \|\lambda_{t+1} - \lambda\|^2) - \frac{\eta}{2}\|\nabla_\lambda \mathcal{L}_t(\mathbf{x}_t, \lambda_t)\|^2$$

$$= -\frac{1}{2\eta}(\|\lambda_t - \lambda\|^2 - \|\lambda_{t+1} - \lambda\|^2) - \frac{\eta}{2}\| - \widetilde{g}_t(\mathbf{x}_t) + \delta\eta\lambda_t\|^2$$

$$\text{(from (46))}$$

$$= -\frac{1}{2\eta}(\|\lambda_t - \lambda\|^2 - \|\lambda_{t+1} - \lambda\|^2) - \frac{\eta}{2}\| - \widehat{g}_t(\mathbf{x}_t) - \gamma_t + \delta\eta\lambda_t\|^2$$

$$\geq -\frac{1}{2\eta}(\|\lambda_t - \lambda\|^2 - \|\lambda_{t+1} - \lambda\|^2) - \eta\|\widehat{g}_t(\mathbf{x}_t)\|^2 - 2\eta\gamma_t^2 - 2\delta^2\eta^3\lambda_t^2$$

$$\text{(apply } \|a + b\|^2 \leq 2\|a\|^2 + 2\|b\|^2 \text{ twice)}$$

$$\geq -\frac{1}{2\eta}(\|\lambda_t - \lambda\|^2 - \|\lambda_{t+1} - \lambda\|^2) - C^2\eta - 2\eta\gamma_t^2 - 2\delta^2\eta^3\lambda_t^2, \quad (53)$$

where the last inequality follows from the definition of $C := \max_{\mathbf{p}' \sim \mathcal{D}_p} \max_{\mathbf{x} \in \mathcal{K}} |\langle \mathbf{p}', \mathbf{x}\rangle - \frac{B_T}{T}|$ and $\widehat{g}_t(\mathbf{x}) := \langle \widehat{\mathbf{p}}_t, \mathbf{x}\rangle - \frac{B_T}{T}$. From convexity of function $\mathcal{L}_t(\mathbf{x}, \lambda)$ w.r.t $\lambda$, we have

$$\mathcal{L}_t(\mathbf{x}_t, \lambda) - \mathcal{L}_t(\mathbf{x}_t, \lambda_t) \geq (\lambda - \lambda_t)^\top \nabla_\lambda \mathcal{L}_t(\mathbf{x}_t, \lambda_t)$$

$$\geq -\frac{1}{2\eta}(\|\lambda_t - \lambda\|^2 - \|\lambda_{t+1} - \lambda\|^2) - C^2\eta - 2\eta\gamma_t^2 - 2\delta^2\eta^3\lambda_t^2 \quad (54)$$

where (54) follows from (53).

Subtracting (54) from (51), we get

$$\mathbb{E}[(1 - 1/e)\mathcal{L}_t(\mathbf{x}, \lambda_t) - \mathcal{L}_t(\mathbf{x}_t, \lambda)] \leq \frac{1}{2\eta}\mathbb{E}[\|\mathbf{x}_t - \mathbf{x}\|^2 - \|\mathbf{x}_{t+1} - \mathbf{x}\|^2] + \frac{1}{2\eta}(\|\lambda_t - \lambda\|^2 - \|\lambda_{t+1} - \lambda\|^2)$$

$$+ G_F^2\eta + \eta\beta^2\lambda_t^2 + C^2\eta + 2\eta\gamma_t^2 + 2\delta^2\eta^3\lambda_t^2 + \frac{1}{e}\lambda_t\widetilde{g}_t(\mathbf{x}) - \frac{\delta\eta}{2e}\lambda_t^2.$$

$$(55)$$

Summing (55) for $t \in [T]$, we have

$$\sum_{t=1}^{T} \mathbb{E}[(1 - 1/e)\mathcal{L}_t(\mathbf{x}, \lambda_t) - \mathcal{L}_t(\mathbf{x}_t, \lambda)]$$

$$\leq \frac{1}{2\eta}\sum_{t=1}^{T}\mathbb{E}[\|\mathbf{x}_t - \mathbf{x}\|^2 - \|\mathbf{x}_{t+1} - \mathbf{x}\|^2] + \frac{1}{2\eta}\sum_{t=1}^{T}(\|\lambda_t - \lambda\|^2 - \|\lambda_{t+1} - \lambda\|^2)$$

$$+ G_F^2\eta T + \eta\beta^2\sum_{t=1}^{T}\lambda_t^2 + C^2\eta T + 2\eta\sum_{t=1}^{T}\gamma_t^2 + 2\delta^2\eta^3\sum_{t=1}^{T}\lambda_t^2 + \frac{1}{e}\sum_{t=1}^{T}\lambda_t\widetilde{g}_t(\mathbf{x}) - \frac{\delta\eta}{2e}\sum_{t=1}^{T}\lambda_t^2$$

$$(56)$$

$$\leq \frac{1}{2\eta}\mathbb{E}[\|\mathbf{x}_1 - \mathbf{x}\|^2 - \|\mathbf{x}_{T+1} - \mathbf{x}\|^2] + \frac{1}{2\eta}(\|\lambda_1 - \lambda\|^2 - \|\lambda_{T+1} - \lambda\|^2)$$

$$+ G_F^2\eta T + \eta\beta^2\sum_{t=1}^{T}\lambda_t^2 + C^2\eta T + 2\eta\sum_{t=1}^{T}\gamma_t^2 + 2\delta^2\eta^3\sum_{t=1}^{T}\lambda_t^2 + \frac{1}{e}\sum_{t=1}^{T}\lambda_t\widetilde{g}_t(\mathbf{x}) - \frac{\delta\eta}{2e}\sum_{t=1}^{T}\lambda_t^2$$

$$(57)$$

$$\leq \frac{1}{2\eta}\mathbb{E}[\|\mathbf{x}_1 - \mathbf{x}_{T+1}\|^2] + \frac{1}{2\eta}(\|\lambda_1 - \lambda\|^2 - \|\lambda_{T+1} - \lambda\|^2)$$

$$+ G_F^2\eta T + \eta\beta^2\sum_{t=1}^{T}\lambda_t^2 + C^2\eta T + 2\eta\sum_{t=1}^{T}\gamma_t^2 + 2\delta^2\eta^3\sum_{t=1}^{T}\lambda_t^2 + \frac{1}{e}\sum_{t=1}^{T}\lambda_t\widetilde{g}_t(\mathbf{x}) - \frac{\delta\eta}{2e}\sum_{t=1}^{T}\lambda_t^2$$

$$\text{(triangle inequality)}$$

$$\leq \frac{d^2}{2\eta} + \frac{\lambda^2}{2\eta} + G_F^2\eta T + \eta\beta^2\sum_{t=1}^{T}\lambda_t^2 + C^2\eta T + 2\eta\sum_{t=1}^{T}\gamma_t^2 + 2\delta^2\eta^3\sum_{t=1}^{T}\lambda_t^2 + \frac{1}{e}\sum_{t=1}^{T}\lambda_t\widetilde{g}_t(\mathbf{x}) - \frac{\delta\eta}{2e}\sum_{t=1}^{T}\lambda_t^2.$$

$$(58)$$

where (58) uses Assumption 1 with def. of diameter $d$,$\lambda_1 = 0$, and $-\|\cdot\|^2 \leq 0$. Expanding the left hand side of (58), we deduce

$$\sum_{t=1}^{T}[(1-1/e)f(\mathbf{x}) - f(\mathbf{x}_t)] + \sum_{t=1}^{T}[\lambda \widetilde{g}_t(\mathbf{x}_t) - (1-1/e)\lambda_t \widetilde{g}_t(\mathbf{x})] + \sum_{t=1}^{T}[(1-1/e)\frac{\delta\eta\lambda_t^2}{2} - \frac{\delta\eta\lambda^2}{2}]$$

$$\leq \frac{d^2}{2\eta} + \frac{\lambda^2}{2\eta} + G_F^2\eta T + \eta\beta^2\sum_{t=1}^{T}\lambda_t^2 + C^2\eta T + 2\eta\sum_{t=1}^{T}\gamma_t^2 + 2\delta^2\eta^3\sum_{t=1}^{T}\lambda_t^2 + \frac{1}{e}\sum_{t=1}^{T}\lambda_t\widetilde{g}_t(\mathbf{x}) - \frac{\delta\eta}{2e}\sum_{t=1}^{T}\lambda_t^2.$$
(59)

Rearranging, we have

$$\sum_{t=1}^{T}[(1-1/e)f(\mathbf{x}) - f(\mathbf{x}_t)] + \left\{\lambda\sum_{t=1}^{T}\widetilde{g}_t(\mathbf{x}_t) - \left(\frac{\delta\eta T}{2} + \frac{1}{2\eta}\right)\lambda^2\right\}$$

$$\leq \sum_{t=1}^{T}\lambda_t\widetilde{g}_t(\mathbf{x}) + \eta\left(\beta^2 + 2\delta^2\eta^2 - \frac{\delta}{2}\right)\sum_{t=1}^{T}\lambda_t^2 + \frac{d^2}{2\eta} + G_F^2\eta T + C^2\eta T + 2\eta\sum_{t=1}^{T}\gamma_t^2. \quad (60)$$

Set $\mathbf{x} = \mathbf{x}^*$; From Lemma 4, with probability at least $1 - \frac{\varepsilon}{T}$, $\widetilde{g}_t(\mathbf{x}^*) = \widehat{g}_t(\mathbf{x}^*) - \gamma_t \leq g(\mathbf{x}^*)$ holds. since $\mathbf{x}^*$ satisfies the long term constraint, we have $g(\mathbf{x}^*) \leq 0$. Now we choose $\delta$ such that $\beta^2 + 2\delta^2\eta^2 - \frac{\delta}{2} \leq 0$. This is a quadratic term w.r.t. $\delta$. It is easy to verify that the quadratic formula has real roots when $\eta \leq \frac{\sqrt{2}}{8\beta}$. As we choose $\eta = \frac{d}{U\sqrt{T}}$, this will give us the condition that $T$ needs to be sufficiently large, i.e., $T \geq \frac{32d^2\beta^2}{U^2}$. We can simply choose $\delta = 4\beta^2$.

Applying both of those inequalities to (60) and by union bound, we get with probability at least $1 - \varepsilon$,

$$\sum_{t=1}^{T}[(1-1/e)f(\mathbf{x}^*) - f(\mathbf{x}_t)] + \left\{\lambda\sum_{t=1}^{T}\widetilde{g}_t(\mathbf{x}_t) - \left(\frac{\delta\eta T}{2} + \frac{1}{2\eta}\right)\lambda^2\right\} \leq \frac{d^2}{2\eta} + G_F^2\eta T + C^2\eta T + 2\eta\sum_{t=1}^{T}\gamma_t^2.$$
(61)

Maximizing the LHS of (61) with respect to $\lambda$, over the range $[0, +\infty)$, we get a solution of $\lambda = \frac{\left[\sum_{t=1}^{T}\widetilde{g}_t(\mathbf{x}_t)\right]_+}{\delta\eta T/2 + 1/2\eta}$. Plugging this into (61) gives us

$$\sum_{t=1}^{T}[(1-1/e)f(\mathbf{x}^*) - f(\mathbf{x}_t)] + \frac{\left[\sum_{t=1}^{T}\widetilde{g}_t(\mathbf{x}_t)\right]_+^2}{\delta\eta T/2 + 1/2\eta} \leq \frac{d^2}{2\eta} + G_F^2\eta T + C^2\eta T + 2\eta\sum_{t=1}^{T}\gamma_t^2. \quad (62)$$

Let $U = \max\{G_F, C\}$. Choosing $\eta = \frac{d}{U\sqrt{T}}$, we have with probability at least $1 - \varepsilon$,

$$\sum_{t=1}^{T}[(1-1/e)f(\mathbf{x}^*) - f(\mathbf{x}_t)] + \frac{\left[\sum_{t=1}^{T}\widetilde{g}_t(\mathbf{x}_t)\right]_+^2}{\delta\eta T/2 + 1/2\eta}$$

$$\leq \frac{d\max\{G_F, C\}}{2}\sqrt{T} + \frac{G_F^2 d}{\max\{G_F, C\}}\sqrt{T} + \frac{C^2 d}{\max\{G_F, C\}}\sqrt{T} + \frac{4\max\{G_F, C\}d\log\frac{2T}{\varepsilon}}{\sqrt{T}}\sum_{t=1}^{T}\frac{1}{t}$$

$$\leq \frac{d\max\{G_F, C\}}{2}\sqrt{T} + \max\{G_F, C\}d\sqrt{T} + \max\{G_F, C\}d\sqrt{T} + \frac{4\max\{G_F, C\}d\log\frac{2T}{\varepsilon}}{\sqrt{T}}\sum_{t=1}^{T}\frac{1}{t}$$

$$\leq \frac{5dU}{2}\sqrt{T} + \frac{4dU\log\frac{2T}{\varepsilon}}{\sqrt{T}}\sum_{t=1}^{T}\frac{1}{\sqrt{t}} \quad (63)$$

$$= \frac{5dU}{2}\sqrt{T} + 8dU\log\frac{2T}{\varepsilon} \quad (64)$$

$$= \mathcal{O}(T^{1/2}), \quad (65)$$

and dropping the second term on the LHS gives us the desired $(1 - 1/e)$-regret bound.

Next, we establish our constraint violation bound. Since $F_1 := \max_{\mathbf{x} \in \mathcal{K}} |f(\mathbf{x})|$ and $F_2 := \max_{\mathbf{x}, \mathbf{y} \in \mathcal{K}} |f(\mathbf{x}) - f(\mathbf{y})|$, we have

$$|(1 - 1/e)f(\mathbf{x}^*) - f(\mathbf{x}_t)| \le |f(\mathbf{x}^*) - f(\mathbf{x}_t)| + \frac{1}{e}|f(\mathbf{x}_t)| \le \frac{F_1}{e} + F_2, \tag{66}$$

thus, $\sum_{t=1}^{T}[(1 - 1/e)f(\mathbf{x}^*) - f(\mathbf{x}_t)] \ge -(\frac{F_1}{e} + F_2)T$. Plugging back in (64), we have

$$\frac{\left[\sum_{t=1}^{T} \widetilde{g}_t(\mathbf{x}_t)\right]_+^2}{\delta\eta T + 1/\eta} \le \frac{5dU}{2}\sqrt{T} + 8dU \log \frac{2T}{\varepsilon} + (\frac{F_1}{e} + F_2)T. \tag{67}$$

Rearranging and plug in the value of $\eta$, we have

$$\left[\sum_{t=1}^{T} \widetilde{g}(\mathbf{x}_t)\right]_+ \le \sqrt{\left(\frac{5dU}{2}\sqrt{T} + 8dU \log \frac{2T}{\varepsilon} + (\frac{F_1}{e} + F_2)T\right) \cdot \left(\frac{4\beta^2 d}{U} + \frac{U}{d}\right)\sqrt{T}}. \tag{68}$$

Combining (68) with Lemma 5, we get

$$
\begin{aligned}
C_T &\le \sqrt{\left(\frac{5dU}{2}\sqrt{T} + 8dU \log \frac{2T}{\varepsilon} + (\frac{F_1}{e} + F_2)T\right) \cdot \left(\frac{4\beta^2 d}{U} + \frac{U}{d}\right)\sqrt{T}} + r\sum_{t=1}^{T}\|\widehat{p}_t - p\| + \sum_{t=1}^{T}\gamma_t \\
&\le \sqrt{\left(\frac{5dU}{2}\sqrt{T} + 8dU \log \frac{2T}{\varepsilon} + (\frac{F_1}{e} + F_2)T\right) \cdot \left(\frac{4\beta^2 d}{U} + \frac{U}{d}\right)\sqrt{T}} \\
&\qquad + rQ\sigma\sqrt{T \log\left(\frac{2nT}{\varepsilon}\right)} + \sum_{t=1}^{T}\gamma_t \qquad\qquad\qquad\qquad \text{(using Lemma 3)} \\
&\le \sqrt{\left(\frac{5dU}{2}\sqrt{T} + 8dU \log \frac{2T}{\varepsilon} + (\frac{F_1}{e} + F_2)T\right) \cdot \left(\frac{4\beta^2 d}{U} + \frac{U}{d}\right)\sqrt{T}} \\
&\qquad + rQ\sigma\sqrt{T \log\left(\frac{2nT}{\varepsilon}\right)} + 2\sqrt{2TC^2 \log\left(\frac{2T}{\varepsilon}\right)} \\
&= \mathcal{O}(T^{3/4}). \tag{69}
\end{aligned}
$$

where in the last inequality we plug in the definition of $\gamma_t$. This concludes the proof.

$\square$

# F   More Related Works

We discuss more related works in this section.

**Adversarial Constraints** It is worth noting a line of research that emphasizes adversarial constraints. This setting was initially explored in [21], where a simple counterexample highlighted that achieving sub-linear regret against the best fixed benchmark action in hindsight while maintaining sub-linear total constraint violation, is not always possible. Subsequent works addressed this challenge by introducing additional assumptions to the problem setting to derive meaningful results. Specifically, these works not only required the fixed benchmark action to satisfy the long-term constraint but also imposed the restriction that the benchmark satisfies the constraint proportionally over any window of size $W \in [T]$ [18, 30, 31]. Due to these added assumptions, direct comparisons with our work are not applicable.

**Online Monotone Submodular Maximization** In the discrete domain, online monotone submodular set maximization was first studied in [33] in the adversarial setting where they introduced the meta-action technique. More recently, [24] proposed novel algorithms utilizing the Blackwell Approachability framework, showcasing improved regrets (w.r.t. constant terms) in both semi-bandit and bandit feedback settings. As for stochastic submodular maximization in the discrete

domain, [19] investigated the case of semi-bandit feedback, specifically in the form of marginal gains. Additionally, recent works such as [25, 26] have delved into the full-bandit feedback setting. For continuous domains, [9] first investigate the online (stochastic) gradient ascent (OGA) with a $\frac{1}{2}$-regret of $\mathcal{O}(T^{1/2})$. Then, inspired by the meta actions [33], [9] also proposed Frank-Wolfe type algorithm with a $(1-1/e)$-regret of $\mathcal{O}(T^{1/2})$ when exact gradient is available. When only stochastic gradient is available, [7] proposed a variant of Frank-Wolfe algorithm achieving $(1-1/e)$-regret of $\mathcal{O}(T^{1/2})$, but requires $\mathcal{O}(T^{3/2})$ stochastic gradient queries in each time step. In the effort of reducing gradient queries, [41] achieves $(1-1/e)$-regret of $\mathcal{O}(T^{4/5})$ with only one stochastic gradient evaluation each round. Recently, [42] have proposed an auxiliary function to boost the approximation ratio of the online gradient ascent algorithms from $\frac{1}{2}$ to $1-1/e$.

## G  Query Set and Constraint Set

Let $\mathcal{K} \subset \mathbb{R}^d_+$ denote a convex set, with $\mathcal{K}'$ defined as the convex hull of $\mathcal{K} \cup \{\mathbf{0}\}$. For problems involving monotone functions over a general set $\mathcal{K}$, the optimal solution in $\mathcal{K}'$ is the same as that in $\mathcal{K}$ due to monotonicity. However, addressing this extended problem domain within $\mathcal{K}'$ may necessitate evaluating the function across the larger set, which may not always be feasible. In the literature of stochastic DR-submodular maximization without long-term constraints, various algorithms have been developed to accommodate different constraints (general convex set, convex set containing $\mathbf{0}$, downward-closed set, etc.). Notably, in scenarios where queries are restricted solely to the constraint set, no algorithm—neither online nor offline—has demonstrated an ability to surpass a $1/2$ approximation ratio. For an exhaustive discussion on this matter, we defer to [28].

## H  Application Examples: Origin not Included in Constraint Set

We will discuss the following example motivating applications to highlight the importance of general convex regions $\mathcal{K}$ for which the origin is not feasible. For such problems, the best-known approximation ratio is 1/2 [12].

1. **Maximum Coverage [17]** Imagine a city's emergency management agency aiming to optimize the deployment of emergency response teams (firefighters, medical personnel, rescue teams) in many rounds during crises like major fires or earthquakes. The objective is to maximize coverage across different city zones. Allocating 0 teams to any zone isn't feasible, as it means no emergency response. The goal is to allocate resources in each round to maximize overall response coverage while satisfying the long-term agency budget.

2. **Budget Allocation [32]** Let a bipartite graph $G = (S, T; W)$ represent a social network, where $S$ and $T$ are collections of advertising channels and customers, respectively. The edge weight represents the influence probability of channel $s$ to customer $t$. The goal is to distribute the per-round budget among the source nodes, and to maximize the expected influence on the potential customers over multiple rounds, while satisfying a long-term constraint (e.g., money spent). Corporate management may require a minimum level of customer engagement with each campaign overall or from select target groups. There may also be per-round minimum contractual purchase requirements with the advertising partners. Thus, allocating 0 budget in any round may not be permitted.

3. **Facility location [17]** Consider a scenario where a company needs to decide on the locations (virtual) to set up new service centers to maximize service coverage over multiple rounds while satisfying a total budget constraint over all rounds. At each round, each customer segment must receive at least a certain level of service or coverage, which means 0 is not a feasible solution because no facilities can provide no service.

All the problems above were initially studied in the discrete domain and extended to the continuous domain in [6]. Furthermore, when faced with a discrete objective, one can always use the "relax and rounding" strategy to transition from addressing a discrete problem to tackling a continuous one. Such techniques are widely frequently utilized within the submodular maximization community, as exemplified by the work of [8].

